# Worst-Case Offline Reinforcement Learning with Arbitrary Data Support

**Kohei Miyaguchi**[*]
IBM Research – Tokyo
Tokyo, Japan
koheimiyaguchi@gmail.com

## Abstract

We propose a method of offline reinforcement learning (RL) featuring the performance guarantee *without* any assumptions on the data support. Under such conditions, estimating or optimizing the conventional performance metric is generally infeasible due to the distributional discrepancy between data and target policy distributions. To address this issue, we employ *a worst-case policy value* as a new metric and constructively show that the sample complexity bound of $O(\epsilon^{-2})$ is attainable without any data-support conditions, where $\epsilon > 0$ is the policy suboptimality in the new metric. Moreover, as the new metric generalizes the conventional one, the algorithm can address standard offline RL tasks without modification. In this context, our sample complexity bound can be seen as a strict improvement on the previous bounds under the single-policy concentrability and the single-policy realizability.

## 1 Introduction

Offline reinforcement learning (RL) (Levine et al., 2020; Prudencio et al., 2023) is a framework for learning decision-making policies while constrained to a fixed batch of data, preventing the learner from acquiring new information about the environment during training.

The primary challenges of offline RL are thus originated from the discrepancy between the state-action distribution of the batch data $\mu(s)\beta(a|s)$ and the visitation distribution of the trained policy $d^\pi(s)\pi(a|s)$. Most of the previous studies have avoided directly dealing with this discrepancy by posing the assumption known as *concentrability* (Munos and Szepesvári, 2008; Antos et al., 2008; Chen and Jiang, 2019; Xie et al., 2022). Roughly speaking, the condition asserts that the ratio between these two distributions $d^\pi\pi/\mu\beta$ is well-defined and uniformly bounded over the entire state-action space. This, in turn, constrains the trained policy $\pi$ to strictly stay inside the state space covered by the data support.

However, concetrability may be impractical in real-world applications for several reasons. First, one often ends up with a poor coverage of the state-action space when exhaustive data collection is expensive or practically infeasible as in the domains of autonomous driving (Fang et al., 2022), healthcare (Yu et al., 2021) and public policy-making (Abe et al., 2010). Moreover, the precise shape of the partial coverage is unknown if the considerations making it partial are not well-documented or disclosed. On the other hand, it is generally difficult to accurately predict if a policy will visit a given state or not based only on the knowledge of the policy and the batch dataset. As a result, the set of concentrable policies in a hypothesis space may be too small to achieve reasonable performance or

---

[*]The author is affiliated with LY Corporation at the time of publication.

Table 1: **Assumptions and sample complexity bounds of related work.** $\pi^*$ and $\tilde{\pi}^*$ denote optimal policies in the conventional and worst-case offline RL, respectively. $\pi_n$ denotes a sequence of policies indexed with the sample size $n$. The realizability of $\pi$ means that $\pi$-associated model-free parameters (e.g., value functions, visitation weight functions and the policy itself) are realizable. $\epsilon > 0$ is the policy suboptimality given in Problem 4.1 (or equivalently in Problem 3.1, see Corollary 4.2 for the equivalence). $0 < \delta < 1$ denotes the confidence parameter. $H$ denotes the time horizon and roughly comparable to $(1-\gamma)^{-1}$. $C_{\mathrm{gap}}$ and $\beta_{\mathrm{gap}}$ denote the minimum and the lower-tail exponent of the action value gaps, respectively. $\mathcal{N}$ denotes the cardinality of the function classes. The improvements made by our result are emphasized. See Appendix A for more details.

| Method | Assumptions | | Sample complexity bound |
| --- | --- | --- | --- |
| | Concentrability | Realizability | |
| Zhan et al. (2022) | $\pi^*$ | $\pi_n$ | $\epsilon^{-6}(1-\gamma)^{-4}\ln(\mathcal{N}/\delta)$ |
| Chen and Jiang (2022) | $\pi^*$ | $\pi^*$ | $\epsilon^{-2}H^5 C_{\mathrm{gap}}^{-2}\ln(\mathcal{N}/\delta)$ |
| Ozdaglar et al. (2023) | $\pi^*$ | $\pi^*$ | $\epsilon^{-2}(1-\gamma)^{-6}C_{\mathrm{gap}}^{-2}\ln(\mathcal{N}/\delta)$ |
| Uehara et al. (2023) | $\pi^*$ | $\pi^*$ | $\epsilon^{-2-4/\beta_{\mathrm{gap}}}(1-\gamma)^{-6-4/\beta_{\mathrm{gap}}}\ln(\mathcal{N}/\delta)$ |
| Ours (Corollary 6.3) | — | $\tilde{\pi}^*$ | $\epsilon^{-2}(1-\gamma)^{-4}\ln(\mathcal{N}/\delta)$ |

even empty.[2] Therefore, for applying offline RL in such domains, we need a method that works well without concentrability or any coverage-related conditions.

To tackle with this issue, we study offline RL with arbitrary data support. We present two major results in this paper.

i) We develop *worst-case offline RL* (Problem 4.1), a new offline RL framework for handling poor state-action coverage, which can be seen as a natural generalization of conventional offline RL (Corollary 4.2).

ii) We develop *worst-case minimax RL* (WMRL, Section 6.3), a model-free algorithm addressing worst-case offline RL (Corollary 6.3). The resulting sample complexity bound improves the previous state of the art in terms of both the weakness of the assumptions and the strength of the bound (Table 1).

The rest of the paper is organized as follows. In Section 2, we review the previous work in the literature of offline RL, centered around theoretical studies on the role of concentrability. In Section 3, we introduce some preliminaries around Markov decision process (MDP), offline RL and concentrability. Then, in Section 4, with the observation that offline RL is ill-posed without concentrability, we introduce *worst-case offline RL* as a natural generalization and discuss some of its properties useful in our subsequent analysis. In Section 5, we establish the connection between worst-case offline RL and the Lagrangians derived from the saddle-point formulation of offline RL. In Section 6, exploiting the connection established earlier, we construct a method for solving worst-case offline RL with polynomial sample complexity. Finally, we discuss the limitation and the future work in Section 7.

## 2 Related Work

The notion of concentrability is introduced by Munos (2003); Munos and Szepesvári (2008); Antos et al. (2008) to analyze the value/policy iteration algorithms, not necessarily in the context of offline RL. Recently, it has been increasingly gaining traction as one of the key characteristics of the difficulty of offline RL (Chen and Jiang, 2019) due to the distribution mismatch. In its original definition, concentrability requires the norm of the density ratio $\|d^\pi\pi/\mu\beta\|_\infty$ to be bounded *uniformly* for all the policies $\pi$. Liu et al. (2020) showed that this uniform boundedness can be relaxed to the single-policy boundedness with the principle of *pessimism in the face of uncertainty (PFU)*. Considering the case where the single-policy concentrability is even slightly violated, Xie et al. (2021) further analyzed the

performance degradation caused by the lack of concentrability. Finally, we completely remove the concentrability assumption by incorporating it into a new performance metric.

The removal of concentrability is useful not only for widening the applicability of offline RL, but also strengthening the sample complexity bound by streamlining the analysis. Previously, Zhan et al. (2022) established polynomial sample complexity bounds under the weakest known model-free assumptions, yet being unable to achieving the statistically reasonable rate $O(\epsilon^{-2})$. Also, Chen and Jiang (2022); Ozdaglar et al. (2023); Uehara et al. (2023) gave improved rates additionally assuming that the minimum action value gap is bounded away from zero. On the other hand, under a set of assumptions as weak as Zhan et al. (2022), our sample complexity bound achieves the rate of $O(\epsilon^{-2})$. See Appendix A for more detailed discussions.

Algorithmically, the PFU principle is often materialized as the pessimistic or behavioral regularization (Kumar et al., 2020; Fujimoto and Gu, 2021; Yu et al., 2020). Previous analyses are often sensitive to the hyperparameters controlling the degree of such regularization, such as the truncation threshold $b$ in Liu et al. (2020), the Bellman consistency threshold $\varepsilon$ in Xie et al. (2021); Chen and Jiang (2022) and the regularization weight in Zhan et al. (2022); Uehara et al. (2023). On the other hand, our method has no hyperparameter other than the choice of the function approximators. One may see the root cause of this difference in that the PFU principle is built into our single new performance metric, whereas the previous studies adopt it as an additional objective, resulting in bi-objective optimizations.

## 3 Preliminaries

We denote the set of nonnegative real numbers by $\mathbb{R}_+ = [0, \infty)$ and the uniform norm of a function $g$ over its domain by $\|g\|_\infty := \sup_{z \in \mathrm{dom}(g)} |g(z)|$. We also denote by $\Delta(\mathcal{X})$ the set of (generalized) probability density functions on $\mathcal{X}$ relative to a suitable base measure,[3] such as the counting measure and the Lebesgue measure.

**Markov decision process (MDP) and RL.** Let $\mathcal{M} = (\mathcal{S}, \mathcal{A}, R, T)$ be an MDP consisting of the state space $\mathcal{S}$, the action space $\mathcal{A}$, the reward function $R : \mathcal{S} \times \mathcal{A} \to \Delta([0, 1])$ and the transition probability $T : \mathcal{S} \times \mathcal{A} \to \Delta(\mathcal{S})$. We assume both $\mathcal{S}$ and $\mathcal{A}$ are finite sets for simplicity. The goal of RL in general is to optimizing policy $\pi : \mathcal{S} \to \Delta(\mathcal{A})$ in terms of the *policy value*,

$$J(\pi) = J(\pi|\mathcal{M}) := (1 - \gamma)\mathbb{E}^\pi \left[ \sum_{t \geq 0} \gamma^t r_t \right],$$

with a discount factor $\gamma \in (0, 1)$. Here, the expectation $\mathbb{E}^\pi$ is taken with respect to the Markov chain generated with $a_t \sim \pi(\cdot|s_t)$, $r_t \sim R(s_t, a_t)$ and $s_{t+1} \sim T(s_t, a_t)$, $t \geq 0$, starting from a known initial-state distribution $s_0 \sim p_0(s)$.

**Offline constraint.** In maximizing $J(\pi)$, the *offline constraint* prohibits us to access the environment $\mathcal{M}$ except through the *offline dataset* $\mathcal{D} := \{(s_i, a_i, r_i, s_i')\}_{i=1}^n$. We assume the dataset is sampled from a fixed distribution $p_{\mathrm{data}}^\mathcal{M} \in \Delta(\mathcal{S} \times \mathcal{A} \times [0, 1] \times \mathcal{S})^n$ such that

$$p_{\mathrm{data}}^\mathcal{M}(\mathcal{D}) = \prod_{i=1}^n \mu(s_i)\,\beta(a_i|s_i)\,R(r_i|s_i, a_i)\,T(s_i'|s_i, a_i),$$

where $\mu \in \Delta(\mathcal{S})$ and $\beta : \mathcal{S} \to \Delta(\mathcal{A})$ are the behavior state distribution and the behavior policy, respectively. Typically, $p_{\mathrm{data}}^\mathcal{M}(\mathcal{D})$ represents the distribution of the past observational data. The problem of offline RL is now formally given as follows.

**Problem 3.1** (Offline RL). *Given the offline dataset $\mathcal{D}$ and a small number $\epsilon > 0$, find a policy $\pi$ achieving $J^* - J(\pi) \leq \epsilon$, where $J^* := \max_{\pi:\mathcal{S}\to\Delta(\mathcal{A})} J(\pi)$.*

**Value, visitation and weight functions.** Let $r(s, a) := \mathbb{E}_{y \sim R(s,a)}[y]$ be the expected reward function and $r^\pi(s) := \sum_a r(s, a)\pi(a|s)$ be its marginalization with respect to policy $\pi$. Let $\mathcal{T}, \mathcal{T}^\pi$ and

$\mathcal{T}_*^\pi$ be the raw transition operator, the policy transition operator and its adjoint given by $\mathcal{T}v(s,a) = \sum_{s'} v(s')T(s'|s,a)$, $\mathcal{T}^\pi v(s) = \sum_a \mathcal{T}v(s,a)\pi(a|s)$ and $\mathcal{T}_*^\pi d(s) = \sum_{s',a'} d(s')\pi(a'|s')T(s|s',a')$, respectively. Then, the *state value function* $v^\pi : \mathcal{S} \to \mathbb{R}$, the *action value function* $q^\pi : \mathcal{S} \times \mathcal{A} \to \mathbb{R}$ and the *state visitation distribution* $d^\pi \in \Delta(\mathcal{S})$ are given by

$$v^\pi = (I - \gamma\mathcal{T}^\pi)^{-1}r^\pi, \qquad q^\pi = r + \gamma\mathcal{T}v^\pi, \qquad d^\pi = (1-\gamma)(I - \gamma\mathcal{T}_*^\pi)^{-1}p_0,$$

as well as the *state weight function* $w^\pi : \mathrm{supp}(\mu) \to \mathbb{R}$ and the *action weight function* $f^\pi : \mathrm{supp}(\mu\beta) \to \mathbb{R}$ by

$$w^\pi(s) := \frac{d^\pi(s)}{\mu(s)}, \qquad\qquad f^\pi(s,a) := w^\pi(s)\,\rho^\pi(s,a),$$

where $\mathrm{supp}(g) := \{x \in \mathrm{dom}(g) \,|\, g(x) \neq 0\}$ denotes the support of function $g$ and $\rho^\pi(s,a) := \frac{\pi(a|s)}{\beta(a|s)}$ is the density ratio of $\pi$ to $\beta$. We also define the optimal value functions by $v^*(s) := \max_\pi v^\pi(s)$ and $q^*(s,a) = \max_\pi q^\pi(s,a)$ as well as the set of the optimal policies $\Pi^* := \{\pi : \mathcal{S} \to \Delta(\mathcal{A}) : v^\pi = v^*\}$, which by definition all attain $J^*$. See Table 2 for the summary of the notation introduced above.

**Concentrability.** A policy $\pi$ is said to be *concentrable* (or satisfying concentrability) if its state-action visitation is contained in the data support, $\mathrm{supp}(d^\pi\pi) \subset \mathrm{supp}(\mu\beta)$. We denote the set of all the concentrable policies by $\Pi_{\mathrm{CC}} := \{\pi : \mathcal{S} \to \Delta(\mathcal{A}) : \mathrm{supp}(d^\pi\pi) \subset \mathrm{supp}(\mu\beta)\}$.

# 4 Worst-Case Offline Reinforcement Learning

In offline RL (Problem 3.1), the information on $\mathcal{M}$ is restricted by the data support $\mathrm{supp}(\mu\beta)$. In such situations, one cannot know about the transition probability $T(s,a)$ and the reward probability $R(s,a)$ for $(s,a) \notin \mathrm{supp}(\mu\beta)$. Consequently, the accurate estimation of $J(\pi)$ is infeasible (even with $n = \infty$) for unconcentrable policies, and thus previous analyses on Problem 3.1 often require that there exists at least one concentrable optimal policy, i.e., $\Pi_{\mathrm{CC}} \cap \mathrm{argmax}_\pi J(\pi) \neq \emptyset$.

To remove such dependency on concentrability, we introduce a performance metric alternative to $J(\pi)$. Let $\mathfrak{U} := \{\mathcal{M}' : p_{\mathrm{data}}^{\mathcal{M}'} = p_{\mathrm{data}}^{\mathcal{M}}\}$ be the set of the environments indistinguishable from the true environment $\mathcal{M}$ with respect to the resulting data distribution $p_{\mathrm{data}}^{\mathcal{M}}$. Noting that $\mathfrak{U}$ is the information-theoretic limit of the uncertainty on $\mathcal{M}$ under the offline constraint, we follow the *pessimism-in-the-face-of-uncertainty* principle and consider the worst case within $\mathfrak{U}$,

$$\tilde{J}(\pi) := \inf_{\mathcal{M}' \in \mathfrak{U}} J(\pi|\mathcal{M}'), \tag{1}$$

which we refer to as *worst-case policy value*. Replacing $J(\pi)$ with $\tilde{J}(\pi)$ in Problem 3.1, we arrive at the following problem.

**Problem 4.1** (Worst-case offline RL). *Given the offline dataset $\mathcal{D}$ and a small number $\epsilon > 0$, find a policy $\pi$ achieving $\tilde{J}^* - \tilde{J}(\pi) \leq \epsilon$, where $\tilde{J}^* := \max_{\pi:\mathcal{S}\to\Delta(\mathcal{A})} \tilde{J}(\pi)$.*

To facilitate the subsequent analysis on Problem 4.1, we next introduce the notion of *truncated environment*, which is similar to, yet different from those previously considered by Liu et al. (2020); Yin and Wang (2021) as it is based on the *true and unknown* data support rather than the empirical one. The truncation is useful for characterizing the worst-case policy value $\tilde{J}(\pi)$.

**Definition 4.1** (Truncated environment). *The truncation of $\mathcal{M}$ with respect to $\mu$ and $\beta$ is given by $\tilde{\mathcal{M}} = (\tilde{\mathcal{S}}, \mathcal{A}, \tilde{T}, \tilde{R})$, where $\tilde{\mathcal{S}} = \mathcal{S} \cup \{\bot\}$ with $\bot$ being an absorbing state with reward zero and*

$$\tilde{R}(r|s,a) := \chi_{\mu,\beta}(s,a)\,R(r|s,a) + (1 - \chi_{\mu,\beta}(s,a))\,\delta_0(r), \tag{2}$$

$$\tilde{T}(s'|s,a) := \chi_{\mu,\beta}(s,a)\,T(s'|s,a) + (1 - \chi_{\mu,\beta}(s,a))\,\delta_\bot(s'), \tag{3}$$

*for $s \in \mathcal{S}$ and $a \in \mathcal{A}$, where $\chi_{\mu,\beta}(s,a) := \mathbb{I}\{\mu(s) > 0\}\,\mathbb{I}\{\beta(a|s) > 0\}$ is the indicator function of the support of $\mu(s)\,\beta(a|s)$ and $\delta_x \in \Delta(\mathcal{X})$ is the Dirac's delta function located at $x \in \mathcal{X}$.*

**Theorem 4.1.** *We have $\tilde{J}(\pi) = J(\pi|\tilde{\mathcal{M}})$ for all $\pi : \mathcal{S} \to \Delta(\mathcal{A})$.*

*Proof (sketch).* It suffices to show $J(\pi|\tilde{\mathcal{M}}) \leq \tilde{J}(\pi) \leq J(\pi|\tilde{\mathcal{M}})$, where the first inequalilty follows from $J(\pi|\tilde{\mathcal{M}}) \leq J(\pi|\mathcal{M}')$ for all $\mathcal{M}' \in \mathfrak{U}$ and the second inequality follows from $\tilde{\mathcal{M}} \in \mathfrak{U}$. See Appendix D.1 for the complete proof. □

In other words, worst-case offline RL is nothing but offline RL with the truncated environment $\tilde{\mathcal{M}}$. Thus, in principle, one can exploit conventional offline RL methods to solve Problem 4.1. We hereafter refer to the truncated counterparts (those defined by replacing $\mathcal{M}$ with $\tilde{\mathcal{M}}$) of $v^\pi$, $v^*$, $q^\pi$, $q^*$, $\Pi^*$, $d^\pi$, $w^\pi$ and $f^\pi$ as $\tilde{v}^\pi$, $\tilde{v}^*$, $\tilde{q}^\pi$, $\tilde{q}^*$, $\tilde{\Pi}^*$, $\tilde{d}^\pi$, $\tilde{w}^\pi$ and $\tilde{f}^\pi$, respectively. Likewise, let $\tilde{r}$, $\tilde{r}^\pi$, $\tilde{\mathcal{T}}$, $\tilde{\mathcal{T}}^\pi$ and $\tilde{\mathcal{T}}_*^\pi$ be the truncated counterparts of $r$, $r^\pi$, $\mathcal{T}$, $\mathcal{T}^\pi$ and $\mathcal{T}_*^\pi$, respectively.

Let us remark several key implications of Theorem 4.1. First, since the unknown parameters $\tilde{R}$ and $\tilde{T}$ of the truncated environment $\tilde{\mathcal{M}}$ are only nontrivial on the data support, it is intuitively obvious that $\tilde{J}(\pi)$ can be accurately estimated even without the concetrability, given sufficiently large $n$. Thus, it is reasonable to expect that Problem 4.1 does not require any concetrabilities to be well-posed, unlike Problem 3.1. Second, the constructive existence of $\tilde{\mathcal{M}}$ makes the relationship between $J(\pi)$ and $\tilde{J}(\pi)$ clearer, as stated in the following corollary.

**Corollary 4.1.** *We have $\tilde{J}(\pi) = J(\pi)$ if $\pi \in \Pi_{CC}$ and $\tilde{J}(\pi) \leq J(\pi)$ otherwise.*

*Proof.* See Appendix D.2. □

According to Corollary 4.1, the pessimism introduced by the truncation is mild in the sense that it conserves the values of concentrable policies. Finally, it also clarifies the relationship between the suboptimality metrics of the conventional and the worst-case problems.

**Corollary 4.2.** *For all $\pi : \mathcal{S} \to \Delta(\mathcal{A})$, we have*

$$J^* - J(\pi) \leq \tilde{J}^* - \tilde{J}(\pi) \tag{4}$$

*if $\Pi_{CC} \cap \operatorname{argmax}_\pi J(\pi) \neq \emptyset$. Moreover, the equality is attained if in addition $\pi \in \Pi_{CC}$.*

*Proof.* Trivial from Corollary 4.1. □

In other words, solutions of the worst-case problem are also valid as solutions of the conventional problem under the standard assumption, while the two solution concepts are identical if only concentrable policies are concerned. In this sense, worst-case offline RL is a natural generalization of the conventional offline RL for handling arbitrary data distributions.

Finally, we conclude this section by showing a useful property of the worst-case optimal policies. Let $\Pi_\beta := \{\pi : \mathcal{S} \to \Delta(\mathcal{A}) \mid \operatorname{supp}(\mu\pi) \subset \operatorname{supp}(\mu\beta)\}$ be the set of the *on-support* policies, i.e., the policies with the support covered by the behavior policy. The following lemma allows us to limit the scope of policy optimization to $\Pi_\beta$ without sacrificing the optimality in terms of $\tilde{\mathcal{M}}$. The proof is relegated to Appendix D.3.

**Lemma 4.1.** *There is at least one worst-case optimal policy that is on-support, i.e., $\tilde{\Pi}^* \cap \Pi_\beta \neq \emptyset$.*

## 5 Lagrangians for Worst-Case Offline Reinforcement Learning

In this section, we set up theoretical foundation of worst-case offline RL. Specifically, in Section 5.1, we show a connection between $\tilde{J}(\pi)$ and the Lagrangian of RL (Puterman, 2014). However, since the Lagrangian in its original form is unstable to the function approximation error (Section 5.2), we further introduce a regularized variant of it (Section 5.3).

### 5.1 Unregularized Lagrangian

Consider the following functional of $v : \mathcal{S} \to \mathbb{R}_+$ and $f : \operatorname{supp}(\mu\beta) \to \mathbb{R}_+$,

$$L(v, f) := (1 - \gamma)\mathbb{E}_{p_0}[v(s)] + \mathbb{E}_{\mu, \beta}\left[f(s, a)\,\delta^{\mathrm{TD}}v(s, a)\right], \tag{5}$$

where $\mathbb{E}_{p_0}$ and $\mathbb{E}_{\mu,\beta}$ are the expectation operators with respect to $s \sim p_0(s)$ and $(s,a) \sim \mu(s)\beta(a|s)$, respectively, and $\delta^{\mathrm{TD}} : \mathbb{R}^{\mathcal{S}} \to \mathbb{R}^{\mathcal{S} \times \mathcal{A}}$ is the time-difference error operator given by $\delta^{\mathrm{TD}} v(s,a) = r(s,a) + \gamma \mathcal{T} v(s,a) - v(s)$.

We refer to $L(v,f)$ as *the (unregularized) Lagrangian* since it has been known as the Lagrangian of the linear-programming-based formulations of RL (Puterman, 2014; Chen and Wang, 2016; Nachum et al., 2019; Zhang et al., 2021; Zhan et al., 2022). The following theorem reveals that, perhaps surprisingly, it is also connected with worst-case offline RL.

**Theorem 5.1.** *For all $\pi \in \Pi_\beta \cap \tilde{\Pi}^*$, $(\tilde{v}^*, \tilde{f}^\pi)$ is a saddle point of $L(v,f)$ in $\mathbb{R}_+^{\mathcal{S}} \times \mathbb{R}_+^{\mathcal{S} \times \mathcal{A}}$.*

*Proof (sketch).* The key of the proof is the following identity of Lagrangian.

**Lemma 5.1.** *For all $\pi \in \Pi_\beta \cap \tilde{\Pi}^*$, we have*

$$L(v,f) = \tilde{J}^* - \mathbb{E}_{\mu,\beta} \left[ (f - \tilde{f}^\pi)(s,a)\,(I - \gamma\tilde{\mathcal{T}})(v - \tilde{v}^*)(s,a) \right] + D_{\mathrm{V}}^\pi(v) - D_{\mathrm{F}}^*(f), \quad (6)$$

*where $D_{\mathrm{V}}^\pi(v) := \sum_{s \notin \mathrm{supp}(\mu)} \tilde{d}^\pi(s)\,v(s)$ and $D_{\mathrm{F}}^*(f) := \mathbb{E}_{\mu,\beta}\left[ f(s,a)\left\{ \tilde{v}^*(s) - \tilde{q}^*(s,a) \right\} \right].$*

Now, observe that the third term $D_{\mathrm{V}}^\pi(v)$ and the fourth term $D_{\mathrm{F}}^*(f)$ in Eq. (6) are nonnegative since all of $v, f, \tilde{d}^\pi$ and $\tilde{v}^* - \tilde{q}^*$ are nonnegative. Therefore, Lemma 5.1 implies that Lagrangian is bounded from below with $L(v,f) \geq \tilde{J}^*$ taking $f = \tilde{f}^\pi$, while bounded from above with $L(v,f) \leq \tilde{J}^*$ taking $v = \tilde{v}^*$. Combining these two inequalities, a class of the saddle points of Lagrangian is identified as desired. The full proof is found in Appendix E.2. $\qquad\square$

Note that the previous studies on the LP-based formulation of RL typically consider the saddle points in a different domain, $\mathbb{R}^{\mathcal{S}} \times \mathbb{R}_+^{\mathcal{S} \times \mathcal{A}}$, where the domain of the primal variable $v$ is unconstrained, unlike our setting with the constraint $v \geq 0$. This constraint is the key to establish the connection with the worst-case environment $\tilde{\mathcal{M}}$.

Theorem 5.1 superficially suggests that finding the saddle points of $L(v,f)$ is a reasonable way of finding the optimal policies with respect to $\tilde{\mathcal{M}}$. However, in the next section, we show that it is unstable and easily breaks down by the function approximation error.

## 5.2 Instability of Unregularized Lagrangian

When the state space $\mathcal{S}$ is large, it is practically infeasible to find a saddle point of $L(v,f)$ naïvely searching over the whole space $\mathbb{R}_+^{\mathcal{S}} \times \mathbb{R}_+^{\mathcal{S} \times \mathcal{A}}$. Therefore, one may introduce compact function classes $\mathcal{V} \subset \mathbb{R}_+^{\mathcal{S}}$ and $\mathcal{F} \subset \mathbb{R}_+^{\mathcal{S} \times \mathcal{A}}$ and limit the scope of the search to these classes. Since we do not know the saddle points (which motivates us to find one), such function classes likely incur the function approximation error. Thus, it is likely that the saddle points $(\tilde{v}^*, \tilde{f}^\pi)$ may sit near the search space $\mathcal{V} \times \mathcal{F}$, but not exactly included in the space, $(\tilde{v}^*, \tilde{f}^\pi) \notin \mathcal{V} \times \mathcal{F}$.

In this context, we show even a tiny function approximation error can completely disrupt the connection established in Theorem 5.1. Consider the function classes $\mathcal{V}_\epsilon = \{v \in \mathbb{R}_+^{\mathcal{S}} \,|\, v \geq \tilde{v}^* + \epsilon\}$ and $\mathcal{F}_\epsilon = \{f \in \mathbb{R}_+^{\mathcal{S} \times \mathcal{A}} \,|\, \exists \pi \in \Pi_\beta \cap \tilde{\Pi}^* \text{ s.t. } f \leq \tilde{f}^\pi - \epsilon\}$ with a small constant $\epsilon > 0$. Then, even though the function approximation error is small ($\epsilon$ for both $\mathcal{V}$ and $\mathcal{F}$ in terms of the $L^\infty$-norm), the saddle point is collapsed to zero under the approximations with $\mathcal{V}_\epsilon$ and $\mathcal{F}_\epsilon$. The proof is relegated to Appendix E.3.

**Corollary 5.1.** *Suppose $\mathcal{V}_\epsilon$ and $\mathcal{F}_\epsilon$ are nonempty. Then, the saddle points of $L(v,f)$ in $\mathcal{V}_\epsilon \times \mathbb{R}_+^{\mathcal{S} \times \mathcal{A}}$ must satisfy $f = 0$. Moreover, the saddle points of $L(v,f)$ in $\mathbb{R}_+^{\mathcal{S}} \times \mathcal{F}_\epsilon$ must satisfy $v = 0$.*

## 5.3 Regularized Lagrangian

Corollary 5.1 shows that the saddle points of Lagrangian cannot be used as a reliable way to find the optimal policies with the function approximation. As a workaround, we introduce *a regularized Lagrangian*,

$$K(v,f) := (1-\gamma)\mathbb{E}_\mu[v(s)] + \mathbb{E}_{\mu,\beta}\left[ f(s,a)\,\delta^{\mathrm{TD}} v(s,a) \right] + \frac{(1-\gamma)^2}{2}\|v\|_{2,\bar{\mu}}^2, \quad (7)$$

where $\bar{\mu} := \mu + \gamma \mathcal{T}_*^\beta \mu$ and $\|v\|_{p,\bar{\mu}} := \{\sum_s \bar{\mu}(s) v^p(s)\}^{1/p}$ denotes the $L^p(\bar{\mu})$-norm of the functions over $\mathcal{S}$. Then, it is shown that the regularized Lagrangian is also connected with worst-case offline RL through its saddle points. To see this, let us define the regularized counterparts of $\check{d}^\pi$, $\check{w}^\pi$ and $\check{f}^\pi$ with $\check{d}^\pi := (1-\gamma)(I - \gamma \tilde{\mathcal{T}}_*^\pi)^{-1}\check{p}^\pi$, $\check{w}^\pi := \check{d}^\pi / \mu$, $\check{f}^\pi := \check{w}^\pi \rho^\pi$, obtained by substituting the initial state distribution $p_0$ with $\check{p}^\pi := \mu + (1-\gamma)\tilde{v}^\pi \bar{\mu}$.

**Theorem 5.2.** *For all $\pi \in \Pi_\beta \cap \tilde{\Pi}^*$, $(\tilde{v}^*, \check{f}^\pi)$ is a saddle point of $K(v,f)$ in $\mathbb{R}_+^\mathcal{S} \times \mathbb{R}_+^{\mathcal{S} \times \mathcal{A}}$. Moreover, the primal solution $\tilde{v}^*$ is unique on $\mathrm{supp}(\bar{\mu})$.*

*Proof (sketch).* Similarly as the proof of Theorem 5.1, the key is the following identity.

**Lemma 5.2.** *There exist $U^* \in \mathbb{R}$ such that, for all $\pi \in \Pi_\beta \cap \tilde{\Pi}^*$,*

$$K(v,f) = U^* - \mathbb{E}_{\mu,\beta}\Big[(f - \check{f}^\pi)(s,a)\,(I - \gamma\tilde{\mathcal{T}})(v - \tilde{v}^*)(s,a)\Big] + \check{D}_{\mathrm{V}}^\pi(v) - D_{\mathrm{F}}^*(f), \qquad (8)$$

*where $\check{D}_{\mathrm{V}}^\pi(v) := \sum_{s \notin \mathrm{supp}(\mu)} \check{d}^\pi(s)\,v(s) + \frac{(1-\gamma)^2}{2}\|v - \tilde{v}^\pi\|_{2,\bar{\mu}}^2$.*

The first claim of Theorem 5.2 then follows from the fact $\check{D}_{\mathrm{V}}^\pi(v)$ is nonnegative, and the second claim follows from the strong convexity of $K(v,f)$ with respect to $v$ on $\mathrm{supp}(\bar{\mu})$. The full proof is in Appendix E.5. □

Comparing Theorems 5.1 and 5.2, it turns out that the regularization does not alter the primal part of the saddle points $\tilde{v}^*$. Moreover, since $K(v,f)$ is strongly convex in terms of $v$, the regularized solution is more stable against the function approximation error as opposed to the unregularized solution. To see this, denote the regularized primal solution under the function approximation by

$$\tilde{v}_\sharp^* \in \operatorname*{argmin}_{v \in \mathcal{V}} \max_{f \in \mathcal{F}} K(v,f), \qquad (9)$$

where $\mathcal{V} \subset \mathbb{R}_+^\mathcal{S}$ and $\mathcal{F} \subset \mathbb{R}_+^{\mathcal{S} \times \mathcal{A}}$ are compact function classes. Also denote the individual function approximation errors of $\mathcal{V}$ and $\mathcal{F}$ by

$$\epsilon_\mathcal{V} := \min_{v \in \mathcal{V}} \|v - \tilde{v}^*\|_{1,\bar{\mu}}, \quad \epsilon_\mathcal{F} := \min_{f \in \mathcal{F}, \pi \in \Pi_\beta} \Big\{ \bar{B}_\mathcal{V}\|f - \check{f}^\pi\|_{1,\mu\beta} + 2(1-\gamma)\|\tilde{v}^\pi - \tilde{v}^*\|_{1,\bar{\mu}} \Big\}, \quad (10)$$

where $\bar{B}_\mathcal{V} := \max\{1 + \gamma B_\mathcal{V}, B_\mathcal{V}\}$ and $B_\mathcal{V} := \max_{v \in \mathcal{V}}\|v\|_\infty$ are scale factors of $\mathcal{V}$. Then, the following lemma shows the stability of the approximate solution $\tilde{v}_\sharp^*$ in terms of the aggregated function approximation error

$$\varepsilon_{\mathrm{app,V}}(\mathcal{V}, \mathcal{F}) := \frac{\sqrt{2\,(2 + B_\mathcal{F})\,\epsilon_\mathcal{V}} + 4\sqrt{\epsilon_\mathcal{F}}}{1 - \gamma} = O\left(\frac{\sqrt{B_\mathcal{F}\,\epsilon_\mathcal{V} + \epsilon_\mathcal{F}}}{1 - \gamma}\right),$$

where $B_\mathcal{F} := \max_{f \in \mathcal{F}}\|f\|_\infty$ is the scale factor of $\mathcal{F}$. The proof is relegated to Appendix E.6.

**Lemma 5.3** (Stability of the regularized primal solution). *We have $\|\tilde{v}_\sharp^* - \tilde{v}^*\|_{2,\bar{\mu}} \le \varepsilon_{app,\mathrm{V}}(\mathcal{V}, \mathcal{F})$.*

Note that the approximation error of $\mathcal{F}$ is trivially bounded by a simpler error term

$$\epsilon_\mathcal{F} \le \bar{B}_\mathcal{V} \min_{f \in \mathcal{F}, \pi \in \Pi_\beta \cap \tilde{\Pi}^*} \|f - \check{f}^\pi\|_{1,\mu\beta},$$

i.e., the $L^1$-error with respect to the function $\check{f}^\pi$ of the *optimal* on-support policy $\pi \in \Pi_\beta \cap \tilde{\Pi}^*$. Our definition of the error is weaker than that, measuring the error with respect to that of the *possibly suboptimal* on-support policy $\pi \in \Pi_\beta$ in exchange for the additional suboptimality cost $2(1-\gamma)\|\tilde{v}^\pi - \tilde{v}^*\|_{1,\bar{\mu}}$. This is beneficial if the optimal policies are difficult to approximate, like deterministic policies in a continuous action space, yet some near-optimal policies such as the soft-optimal policies $\pi(a|s) \propto \exp\{-\eta \tilde{q}^*(s,a)\}$ are easy to approximate.

We also note that the idea of stabilizing the saddle-point-based policy optimization via a strongly convex regularizer is not new (Nachum et al., 2019; Lee et al., 2021; Zhan et al., 2022; Uehara et al., 2023). The major difference here (other than the truncation) is that we regularize the value function $v$ (like Uehara et al. (2023)) but extract the information of the optimal policy from $f$ (like Nachum et al. (2019); Lee et al. (2021); Zhan et al. (2022)), which, combined with our worst-case framework, results in a striking improvement in the sample complexity.

# 6 Worst-Case Minimax Reinforcement Learning

Now, we present a method to solve worst-case offline RL with the saddle points of $K(v, f)$. We first introduce a method of extracting policy from the dual variable $f$ (Section 6.1), then show the suboptimality bound of the extracted policy (Section 6.2), which is our main result, and finally show the sample complexity bound taking into account the finite sample approximation (Section 6.3).

## 6.1 Policy Extraction

Motivated by Theorem 5.2, we propose a method of extracting the worst-case optimal policy $\pi^*$ from the saddle point of $K(v, f)$. Specifically, we consider minimizing the loss function given by

$$D_\Xi(f; w, \pi) := \max_{\xi \in \Xi} \left\{ \mathbb{E}_{\mu,\beta} \left[ f(s, a)\, \xi(s, a) \right] - \mathbb{E}_{\mu,\pi} \left[ \underline{w}(s)\, \xi(s, a) \right] \right\}, \tag{11}$$

where $w : \mathcal{S} \to \mathbb{R}$ is an auxiliary weight function, $\underline{w}(s) := \max\{1 - \gamma, w(s)\}$ is its lower clipping and $\Xi \subset \mathbb{R}^{\mathcal{S} \times \mathcal{A}}$ is a class of discriminator functions. Note that $D_\Xi(f; w, \pi)$ is the integral probability metric (IPM) (Sriperumbudur et al., 2009) between $f(s, a)\, \mu(s)\, \beta(a|s)$ and $\underline{w}(s)\, \mu(s)\, \pi(a|s)$ with respect to the discriminators $\Xi$. With a sufficiently rich $\Xi$, this implies $D_\Xi(f; w, \pi)$ attains its minimum value (i.e., zero) only if $\pi = \pi_f$,[4] thereby informally justifies the minimization of Eq. (11) as a way of policy extraction.

This approach introduces additional (functional) variables to be optimized, $w : \mathcal{S} \to \mathbb{R}$ and $\pi : \mathcal{S} \to \Delta(\mathcal{A})$. To simplify the notation, consider a parameter space $\Theta$ and suppose $f$, $w$ and $\pi$ share the same parameter space $\Theta$, i.e., there exists a mapping $\Theta \ni \theta \mapsto (f_\theta, w_\theta, \pi_\theta)$, and redefine the dual space with $\mathcal{F} = \mathcal{F}(\Theta) := \{f_\theta \,|\, \theta \in \Theta\}$.[5] We define the associated function approximation error with

$$\epsilon_\Theta := \min_{\pi \in \Pi_\beta} \left\{ \bar{B}_{\mathcal{V},\Xi} \epsilon_\Theta(\pi) + 2(1 - \gamma) \|\tilde{v}^* - \tilde{v}^\pi\|_{1,\bar{\mu}} \right\}, \tag{12}$$

where $\bar{B}_{\mathcal{V},\Xi} := \max\{\bar{B}_\mathcal{V}, B_\Xi, \|\tilde{\xi}^*\|_\infty\}$, $B_\Xi := \max_{\xi \in \Xi} \|\xi\|_\infty$ and

$$\epsilon_\Theta(\pi) := \min_{\theta \in \Theta} \left\{ \left\| f_\theta - \check{f}^\pi \right\|_{1,\mu\beta} + \|w_\theta - \check{w}^\pi\|_{1,\mu} + B_{\underline{\mathcal{W}}} \|\pi_\theta - \pi\|_{\mathrm{TV},\mu} \right\} \tag{13}$$

denotes the $\pi$-specific function approximation error of $\Theta$. Here, $B_{\underline{\mathcal{W}}} := \max_{\theta \in \Theta} \|\underline{w}_\theta\|_\infty$ is the boundedness of $\underline{w}_\theta(s)$ and $\|\cdot\|_{\mathrm{TV},\mu}$ is the mean total variation (TV) distance with respect to $\mu$, given by $\|\pi - \pi'\|_{\mathrm{TV},\mu} := \mathbb{E}_\mu \sum_a |\pi(a|s) - \pi'(a|s)|$.

Finally, we conclude this section by introducing key quantities of the policy extraction for the subsequent analysis. Let $B_\Pi := \max_{\theta \in \Theta} \|\pi_\theta/\pi_0\|_\infty$ denote the size of the policy class with respect to some fixed base policy $\pi_0$. Also let $\epsilon_\Xi := \min_{\xi \in \Xi} \|\xi - \tilde{\xi}^*\|_{1,\mu(\beta+\pi_0)}$ be the function approximation error of $\Xi$, where $\tilde{\xi}^*(s, a) := \tilde{q}^*(s, a) - \tilde{v}^*(s)$ is the optimal advantage function.

## 6.2 The Suboptimality Bound

Unifying the saddle-point problem and the policy extraction problem, we arrive at the aggregated loss function

$$\mathcal{L}(\theta) := \mathcal{L}_{\mathrm{SP}}(\theta) + \mathcal{L}_{\mathrm{X}}(\theta), \tag{14}$$

where $\mathcal{L}_{\mathrm{SP}}(\theta) := -\min_{v \in \mathcal{V}} K(v, f_\theta)$ is the loss of $f_\theta$ as a dual solution (cf. Eq. (7)) and $\mathcal{L}_{\mathrm{X}}(\theta) := D_\Xi(f_\theta; w_\theta, \pi_\theta)$ is the loss of the policy extraction from $f_\theta$ to $\pi_\theta$ (cf. Eq. (11)). Let us denote the corresponding estimation error by

$$\epsilon_{\mathrm{est}}(\theta) := \mathcal{L}(\theta) - \min_{\theta \in \Theta} \mathcal{L}(\theta). \tag{15}$$

We also define the aggregated function approximation error with

$$\varepsilon_{\mathrm{app},\Pi}(\mathcal{V}, \Theta, \Xi) := (2 + 3B_\mathcal{F})\, \varepsilon_{\mathrm{app},\mathrm{V}}(\mathcal{V}, \mathcal{F}(\Theta)) + 3\epsilon_\Theta + \{B_\mathcal{F} + (1 - \gamma)B_\Pi\} \epsilon_\Xi. \tag{16}$$

The following theorem establishes an upper bound on the policy suboptimality in terms of $\epsilon_{\mathrm{est}}(\theta)$ and $\varepsilon_{\mathrm{app},\Pi}(\mathcal{V}, \Theta, \Xi)$.

**Theorem 6.1.** *For all $\theta \in \Theta$, we have*

$$\tilde{J}^* - \tilde{J}(\pi_\theta) \le \frac{\|\tilde{w}^{\pi_\theta}\|_\infty}{1 - \gamma} \left\{ \epsilon_{est}(\theta) + \varepsilon_{app,\Pi}(\mathcal{V}, \Theta, \Xi) \right\}. \tag{17}$$

*Proof (sketch).* At the heart of the proof is the following inequalities:

$$\Gamma(\pi_\theta) \le \frac{D_{\mathrm{F}}^*(f_\theta) + \mathcal{L}_{\mathrm{X}}(\theta) + B'\epsilon_\Xi}{1 - \gamma} \le \frac{\epsilon_{est}(\theta) + \varepsilon_{app,\Pi}(\mathcal{V}, \Theta, \Xi)}{1 - \gamma},$$

where $\Gamma(\pi_\theta) := \mathbb{E}_{\mu,\pi_\theta}[\tilde{v}^*(s) - \tilde{q}^*(s,a)]$ denotes the average action value gap with policy $\pi_\theta$ and $B' := B_\mathcal{F} + (1 - \gamma)B_\Pi$. Here, the lower clipping of $\underline{w}$ (Eq. (11)) and the stability of the primal solution (Lemma 5.3) are instrumental in deriving the first and the second inequality, respectively. Then, the proof is completed by bounding $\tilde{J}^* - \tilde{J}(\pi_\theta)$ in terms of $\Gamma(\pi_\theta)$ invoking the performance difference lemma in the worst-case environment. See Appendix F.1 for the full proof. $\square$

Theorem 6.1 suggests that one can minimize the policy suboptimality up to the function approximation error on two conditions, i.e., the weight factor $\|\tilde{w}^{\pi_\theta}\|_\infty$ is appropriately bounded and the loss function $\mathcal{L}(\theta)$ is minimized. In the following, we first discuss how to satisfy the first condition.

A trivial way of bounding $\|\tilde{w}^{\pi_\theta}\|_\infty$ is uniformly bounding it with respect to all $\theta \in \Theta$. Define $\tilde{C}_\infty := \max_{\theta \in \Theta} \|\tilde{w}^{\pi_\theta}\|_\infty$, which we refer to as *the uniform truncated concentrability (UTC) coefficient*. Since $\|\tilde{w}^{\pi_\theta}\|_\infty \le \tilde{C}_\infty$, we get the following simple suboptimality bound.

**Corollary 6.1.** *For all $\theta \in \Theta$, we have*

$$\tilde{J}^* - \tilde{J}(\pi_\theta) \le \frac{\tilde{C}_\infty}{1 - \gamma} \left\{ \epsilon_{est}(\theta) + \varepsilon_{app,\Pi}(\mathcal{V}, \Theta, \Xi) \right\} \tag{18}$$

Note that $\tilde{C}_\infty$ is *always* finite because of the compactness of the whole policy space $\Delta(\mathcal{A})^\mathcal{S}$ and the continuity and well-definedness of $\pi \mapsto \|\tilde{w}^\pi\|_\infty$. Thus, Eq. (18) is non-vacuous for *arbitrary* data distributions as opposed to the conventional concentrability-based results. Moreover, if the conventional bounds are non-vacuous, $\tilde{C}_\infty$ recovers the conventional concentrability coefficient as $\tilde{w}^\pi = w^\pi$.

Eq. (18) can be further refined using the *localized* variants of the uniform coefficient $\tilde{C}_\infty$. The results are presented as Corollaries F.2 and F.3 in Appendix F.3 due to space limitation.

## 6.3 Sample Complexity Analysis

Now that given Corollary 6.1 (and Corollaries F.2 and F.3 as well) bounds the weight factor $\|\tilde{w}^{\pi_\theta}\|_\infty$ with some milder variants of the concentrability coefficient, the remaining task, minimizing $\mathcal{L}(\theta)$, is handled within the framework of the statistical learning, leading to a sample complexity bound. Let

$$\hat{\mathcal{L}}(\theta) := \max_{v \in \mathcal{V}} \max_{\xi \in \Xi} \frac{1}{n} \sum_{z \in \mathcal{D}} \hat{\mathcal{L}}_z(\theta; v, \xi), \tag{19}$$

be the empirical loss function where

$$\hat{\mathcal{L}}_z(\theta; v, \xi) := -\left\{ (1 - \gamma)v(s) + f_\theta(s, a)\{r + \gamma v(s') - v(s)\} + \frac{(1 - \gamma)^2 (v^2(s) + \gamma v^2(s'))}{2} \right\}$$
$$+ f_\theta(s, a)\xi(s, a) - \underline{w}_\theta(s)\mathbb{E}_{a' \sim \pi_\theta(s)}[\xi(s, a')] \tag{20}$$

is the one-sample loss function, $z \equiv (s, a, r, s') \in \mathcal{S} \times \mathcal{A} \times [0, 1] \times \mathcal{S}$ denotes a transition record. Note that Eq. (20) is an unbiased estimator of the objective function $\mathcal{L}(\theta)$.[6] Therefore, it is expected that the oracle loss $\mathcal{L}(\theta)$ can be approximated with the empirical loss $\hat{\mathcal{L}}(\theta)$ and hence the oracle estimation error $\epsilon_{est}(\theta)$ can be approximated with the empirical estimation error

$$\hat{\epsilon}_{est}(\theta) := \hat{\mathcal{L}}(\theta) - \min_{\theta \in \Theta} \hat{\mathcal{L}}(\theta). \tag{21}$$

Formalizing such an intuition, the following corollary shows the empirical counterpart of Corollary 6.1. See Appendix F.5 for the proof.

**Corollary 6.2.** *Let $\mathcal{H} \equiv \mathcal{H}(\mathcal{V}, \Theta, \Xi) \coloneqq \{z \mapsto \hat{\mathcal{L}}_z(\theta; v, \xi) \mid \theta \in \Theta, v \in \mathcal{V}, \xi \in \Xi\}$ be the class of the one-sample loss functions and $\mathfrak{R}_n(\mathcal{H})$ be its Rademacher complexity (Definition B.1). Then, for all $\theta \in \Theta$ and $\delta \in (0, 1)$, with probability $1 - \delta$, we have*

$$\tilde{J}^* - \tilde{J}(\pi_\theta) \leq \frac{\tilde{C}_\infty}{1 - \gamma} \left\{ \hat{\epsilon}_{est}(\theta) + \varepsilon_{app,\Pi}(\mathcal{V}, \Theta, \Xi) + 4\mathfrak{R}_n(\mathcal{H}) + B_{all}\sqrt{\frac{2\ln(2/\delta)}{n}} \right\},$$

*where $B_{all} \coloneqq (1 - \gamma)B_\mathcal{V} + B_\mathcal{F}\bar{B}_\mathcal{V} + (1 - \gamma)^2 B_\mathcal{V}^2 + (B_\mathcal{F} + B_\mathcal{W})B_\Xi$ is the aggregated scale factor.*

The corollary above implies that minimizing Eq. (19), which is possible with the minimax optimizers such as the one developed by Thekumparampil et al. (2019), gives a near-optimal policy in terms of the worst-case environment $\tilde{\mathcal{M}}$, up to the error proportional to the sum of the optimization error $\hat{\epsilon}_{est}(\theta)$, the approximation error $\varepsilon_{app,\Pi}(\mathcal{V}, \Theta, \Xi)$ and the statistical error $O(\mathfrak{R}_n(\mathcal{H}) + n^{-1/2})$. We refer to this method as *worst-case minimax reinforcement learning (WMRL)*.

The next corollary gives the sample complexity of WMRL in the simplified case where the function approximators $\mathcal{V}$, $\Theta$ and $\Xi$ are all finite sets.

**Corollary 6.3.** *Suppose $\mathcal{V}$, $\mathcal{F}$ and $\Xi$ are all finite sets and $\hat{\epsilon}_{est}(\theta) = \varepsilon_{app,\Pi}(\mathcal{V}, \Theta, \Xi) = 0$. Take any $\epsilon > 0$ and $0 < \delta < 1$. Then, we have $\tilde{J}^* - \tilde{J}(\pi_\theta) \leq \epsilon$ with probability $1 - \delta$ if*

$$n = \Omega\left(\frac{B_{all}^2 \tilde{C}_\infty^2}{\epsilon^2(1 - \gamma)^2} \ln \frac{\mathcal{N}}{\delta}\right), \tag{22}$$

*where $\mathcal{N} \coloneqq |\mathcal{V}|\,|\Theta|\,|\Xi|$ denote the product of the cardinalities of the function approximators.*

*Proof.* It follows from Corollary 6.2 with Massart's lemma (Lemma B.2). $\square$

A few remarks follow. First, we can replace the UTC coefficient $\tilde{C}_\infty$ with the localized variants (Definitions F.1 and F.2) to obtain tighter bounds, by making the same argument starting from Corollaries F.2 and F.3 instead of Corollary 6.1, respectively. Second, there are implicit dependencies $B_\mathcal{V} \geq \|\tilde{v}^*\|_\infty$ and $B_\Xi \geq \|\tilde{\xi}^*\|_\infty$ due to the realizability $\tilde{v}^* \in \mathcal{V}$ and $\tilde{\xi}^* \in \Xi$ that follows from $\varepsilon_{app,\Pi}(\mathcal{V}, \Theta, \Xi) = 0$. Hence, Eq. (22) has an implicit $\gamma$-dependency through the scale factor $B_{all}$, which brings an extra $\Theta((1 - \gamma)^{-2})$ factor in the worst case. Table 1 adopts this form for the fairness of comparison.

## 7  Conclusion

To develop an offline RL method for challenging data distributions, we have introduced and studied a generalization of the conventional framework called *worst-case offline RL*. As a result, we have shown it is possible to learn a worst-case optimal policy without any data-support conditions. Moreover, the presented sample complexity bound strictly improves the previous state of the art under the single-policy realizability and the single-policy concentrability, suggesting the utility of the proposed method even with non-challenging data distributions.

We anticipate the presented results are readily extendable to continuous state-action spaces, except that the truncated concentrability coefficients are not unconditionally finite anymore. The results in Appendix F.3 are particularly useful in this context, yet the complete picture on the conditions of their boundedness largely remains to be studied in future work.

## Acknowledgments and Disclosure of Funding

The author is grateful to LY Corporation for providing travel funding.

## Footnotes

[2]There are two typical examples for the empty case: i) when the time horizon of the policy evaluation is (even ever so slightly) longer than the episode length of the collected data in time-inhomogeneous environments, and ii) when there are undocumented restrictions on the actions taken by the behavior policy, but the trained policy is modeled with distributions with full action supports such as Gaussian distributions.

[3]Examples of the *generalized* probability density functions include the Dirac's delta function.

[4] The lower clipping $\underline{w}(s) \geq 1 - \gamma > 0$ plays a crucial role here to exclude the trivial minima $f(s, \cdot) = \underline{w}(s) = 0$ for every $s \in \mathcal{S}$.

[5] There is no loss of generality due to the coupling introduced by the parameter sharing, since one can take it as a product space $\Theta = \Theta_\mathcal{F} \times \Theta_\mathcal{W} \times \Theta_\Pi$.

[6]Note that the expectation with respect to $\pi_\theta$ can be computed exactly since the action space is finite. For the continuous case, one can replace the expectation with the Monte-Carlo approximation without corrupting the unbiasedness.

[7]It is referred to as the *strong* near-optimality since it implies the near-optimality in the usual sense, i.e., $\tilde{J}^* - \tilde{J}(\pi) \leq \epsilon$ for all $\pi \in \tilde{\Pi}_\epsilon^*$.

## References

Abe, N., Melville, P., Pendus, C., Reddy, C. K., Jensen, D. L., Thomas, V. P., Bennett, J. J., Anderson, G. F., Cooley, B. R., Kowalczyk, M., Domick, M., and Gardinier, T. (2010). Optimizing debt

collections using constrained reinforcement learning. In *Proceedings of the 16th ACM SIGKDD International Conference on Knowledge Discovery and Data Mining*, KDD '10, page 75–84, New York, NY, USA. Association for Computing Machinery.

Antos, A., Szepesvári, C., and Munos, R. (2008). Learning near-optimal policies with bellman-residual minimization based fitted policy iteration and a single sample path. *Machine Learning*, 71:89–129.

Chen, J. and Jiang, N. (2019). Information-theoretic considerations in batch reinforcement learning. In *International Conference on Machine Learning*, pages 1042–1051. PMLR.

Chen, J. and Jiang, N. (2022). Offline reinforcement learning under value and density-ratio realizability: the power of gaps. In *Uncertainty in Artificial Intelligence*, pages 378–388. PMLR.

Chen, Y. and Wang, M. (2016). Stochastic primal-dual methods and sample complexity of reinforcement learning. *arXiv preprint arXiv:1612.02516*.

Fang, X., Zhang, Q., Gao, Y., and Zhao, D. (2022). Offline reinforcement learning for autonomous driving with real world driving data. In *2022 IEEE 25th International Conference on Intelligent Transportation Systems (ITSC)*, pages 3417–3422.

Fujimoto, S. and Gu, S. S. (2021). A minimalist approach to offline reinforcement learning. *Advances in neural information processing systems*, 34:20132–20145.

Jiang, N. and Huang, J. (2020). Minimax value interval for off-policy evaluation and policy optimization. In *Proceedings of the 34th International Conference on Neural Information Processing Systems*, NIPS '20, Red Hook, NY, USA. Curran Associates Inc.

Kumar, A., Zhou, A., Tucker, G., and Levine, S. (2020). Conservative q-learning for offline reinforcement learning. *Advances in Neural Information Processing Systems*, 33:1179–1191.

Lee, J., Jeon, W., Lee, B., Pineau, J., and Kim, K.-E. (2021). Optidice: Offline policy optimization via stationary distribution correction estimation. In *International Conference on Machine Learning*, pages 6120–6130. PMLR.

Levine, S., Kumar, A., Tucker, G., and Fu, J. (2020). Offline reinforcement learning: Tutorial, review, and perspectives on open problems. *arXiv preprint arXiv:2005.01643*.

Liu, Y., Swaminathan, A., Agarwal, A., and Brunskill, E. (2020). Provably good batch off-policy reinforcement learning without great exploration. *Advances in neural information processing systems*, 33:1264–1274.

Munos, R. (2003). Error bounds for approximate policy iteration. In *Proceedings of the Twentieth International Conference on International Conference on Machine Learning*, ICML'03, page 560–567. AAAI Press.

Munos, R. and Szepesvári, C. (2008). Finite-time bounds for fitted value iteration. *Journal of Machine Learning Research*, 9(5).

Nachum, O., Dai, B., Kostrikov, I., Chow, Y., Li, L., and Schuurmans, D. (2019). Algaedice: Policy gradient from arbitrary experience. *arXiv preprint arXiv:1912.02074*.

Ozdaglar, A. E., Pattathil, S., Zhang, J., and Zhang, K. (2023). Revisiting the linear-programming framework for offline RL with general function approximation. In *International Conference on Machine Learning*, pages 26769–26791. PMLR.

Prudencio, R. F., Maximo, M. R., and Colombini, E. L. (2023). A survey on offline reinforcement learning: Taxonomy, review, and open problems. *IEEE Transactions on Neural Networks and Learning Systems*.

Puterman, M. L. (2014). *Markov decision processes: discrete stochastic dynamic programming*. John Wiley & Sons.

Rashidinejad, P., Zhu, H., Yang, K., Russell, S., and Jiao, J. (2023). Optimal conservative offline RL with general function approximation via augmented lagrangian. In *The Eleventh International Conference on Learning Representations*.

Shalev-Shwartz, S. and Ben-David, S. (2014). *Understanding machine learning: From theory to algorithms*. Cambridge university press.

Sriperumbudur, B. K., Fukumizu, K., Gretton, A., Schölkopf, B., and Lanckriet, G. R. (2009). On integral probability metrics,\phi-divergences and binary classification. *arXiv preprint arXiv:0901.2698*.

Thekumparampil, K. K., Jain, P., Netrapalli, P., and Oh, S. (2019). Efficient algorithms for smooth minimax optimization. In Wallach, H., Larochelle, H., Beygelzimer, A., d'Alché-Buc, F., Fox, E., and Garnett, R., editors, *Advances in Neural Information Processing Systems*, volume 32. Curran Associates, Inc.

Uehara, M., Kallus, N., Lee, J. D., and Sun, W. (2023). Offline minimax soft-q-learning under realizability and partial coverage. In Oh, A., Naumann, T., Globerson, A., Saenko, K., Hardt, M., and Levine, S., editors, *Advances in Neural Information Processing Systems*, volume 36, pages 12797–12809. Curran Associates, Inc.

Uehara, M. and Sun, W. (2022). Pessimistic model-based offline reinforcement learning under partial coverage. In *International Conference on Learning Representations*.

Xie, T., Cheng, C.-A., Jiang, N., Mineiro, P., and Agarwal, A. (2021). Bellman-consistent pessimism for offline reinforcement learning. In Beygelzimer, A., Dauphin, Y., Liang, P., and Vaughan, J. W., editors, *Advances in Neural Information Processing Systems*.

Xie, T., Foster, D. J., Bai, Y., Jiang, N., and Kakade, S. M. (2022). The role of coverage in online reinforcement learning. *arXiv preprint arXiv:2210.04157*.

Yin, M. and Wang, Y.-X. (2021). Towards instance-optimal offline reinforcement learning with pessimism. In Ranzato, M., Beygelzimer, A., Dauphin, Y., Liang, P., and Vaughan, J. W., editors, *Advances in Neural Information Processing Systems*, volume 34, pages 4065–4078. Curran Associates, Inc.

Yu, C., Liu, J., Nemati, S., and Yin, G. (2021). Reinforcement learning in healthcare: A survey. *ACM Comput. Surv.*, 55(1).

Yu, T., Thomas, G., Yu, L., Ermon, S., Zou, J. Y., Levine, S., Finn, C., and Ma, T. (2020). MOPO: Model-based offline policy optimization. *Advances in Neural Information Processing Systems*, 33:14129–14142.

Zanette, A. (2023). When is realizability sufficient for off-policy reinforcement learning? In *International Conference on Machine Learning*, pages 40637–40668. PMLR.

Zhan, W., Huang, B., Huang, A., Jiang, N., and Lee, J. (2022). Offline reinforcement learning with realizability and single-policy concentrability. In *Conference on Learning Theory*, pages 2730–2775. PMLR.

Zhang, J., Hong, M., Wang, M., and Zhang, S. (2021). Generalization bounds for stochastic saddle point problems. In *International Conference on Artificial Intelligence and Statistics*, pages 568–576. PMLR.

# A   Additional Discussion on Table 1

In Table 1, we compare our result with the previous results on the sample complexity of the conventional offline RL under the weakest known assumptions. The weakest known means that it relies only on the assumptions of the single-policy realizability and the single-policy concentrability. Specifically, they do not assume the Bellman completeness (Munos and Szepesvári, 2008; Xie et al., 2021; Chen and Jiang, 2022) or its variants (Zanette, 2023; Rashidinejad et al., 2023), the model-based realizability (Uehara and Sun, 2022) or the all-policy realizability (Jiang and Huang, 2020), which

| Meaning | Symbol | Type | Definition |
|---|---|---|---|
| Expectation w.r.t. action | $\mathcal{P}^\pi$ | $\mathbb{R}^{\mathcal{S}\times\mathcal{A}} \to \mathbb{R}^{\mathcal{S}}$ | $\mathcal{P}^\pi q(s) = \sum_a q(s,a)\,\pi(a\mid s)$ |
| Extension of action | $\mathcal{P}^\pi_*$ | $\mathbb{R}^{\mathcal{S}} \to \mathbb{R}^{\mathcal{S}\times\mathcal{A}}$ | $\mathcal{P}^\pi_* d(s,a) = d(s)\,\pi(a\mid s)$ |
| Backward transition | $\mathcal{T}$ | $\mathbb{R}^{\mathcal{S}} \to \mathbb{R}^{\mathcal{S}\times\mathcal{A}}$ | $\mathcal{T}v(s,a) = \sum_{s'} v(s')\,T(s'\mid s,a)$ |
|   — with policy | $\mathcal{T}^\pi$ | $\mathbb{R}^{\mathcal{S}} \to \mathbb{R}^{\mathcal{S}}$ | $\mathcal{P}^\pi \mathcal{T}$ |
| Forward transition | $\mathcal{T}_*$ | $\mathbb{R}^{\mathcal{S}\times\mathcal{A}} \to \mathbb{R}^{\mathcal{S}}$ | $\mathcal{T}_* c(s') = \sum_{s,a} T(s'\mid s,a)\,c(s,a)$ |
|   — with policy | $\mathcal{T}^\pi_*$ | $\mathbb{R}^{\mathcal{S}} \to \mathbb{R}^{\mathcal{S}}$ | $\mathcal{T}_* \mathcal{P}^\pi_*$ |
| Marginal reward func. | $r^\pi$ | $\mathcal{S} \to \mathbb{R}$ | $\mathcal{P}^\pi r$ |
| State value func. | $v^\pi$ | $\mathcal{S} \to \mathbb{R}$ | $(I - \gamma\mathcal{T}^\pi)^{-1} r^\pi$ |
|   Optimal — | $v^*$ | $\mathcal{S} \to \mathbb{R}$ | $v^*(s) = \max_\pi v^\pi(s)$ |
| Action value func. | $q^\pi$ | $\mathcal{S} \times \mathcal{A} \to \mathbb{R}$ | $r + \gamma\mathcal{T}v^\pi$ |
|   Optimal — | $q^*$ | $\mathcal{S} \times \mathcal{A} \to \mathbb{R}$ | $q^*(s,a) = \max_\pi q^\pi(s,a)$ |
| Optimal policies | $\Pi^*$ | $2^{\Delta(\mathcal{A})^{\mathcal{S}}}$ | $\{\pi : \mathcal{S} \to \Delta(\mathcal{A}) \,:\, v^\pi = v^*\}$ |
| State visitation dist. | $d^\pi$ | $\mathcal{S} \to \mathbb{R}$ | $(1-\gamma)(I - \gamma\mathcal{T}^\pi_*)^{-1} p_0$ |
| State weight func. | $w^\pi$ | $\mathrm{supp}(\mu) \to \mathbb{R}$ | $d^\pi/\mu$ |
| Policy ratio | $\rho^\pi$ | $\mathrm{supp}(\mu\beta) \to \mathbb{R}$ | $\pi/\beta$ |
| Action weight func. | $f^\pi$ | $\mathrm{supp}(\mu\beta) \to \mathbb{R}$ | $w^\pi\,\rho^\pi$ |
| Policy value | $J(\pi)$ | $\mathbb{R}$ | $(1-\gamma)\mathbb{E}_{p_0}[v^\pi(s)]$ |
|   Optimal — | $J^*$ | $\mathbb{R}$ | $\max_\pi J(\pi)$ |

Table 2: Basic Notation

are strictly stronger than the single-policy realizability and deemed to be rather stringent (Chen and Jiang, 2019). Below, we discuss each of the methods listed in the table.

Leveraging the Lagrangian-based formulation of RL, Zhan et al. (2022) showed the polynomial sample complexity only with the single-policy realizability and the single-policy concentrability for the first time. However, the order of the sample complexity bound $O(\epsilon^{-6})$ is significantly looser than the standard statistical rate $O(\epsilon^{-2})$. Moreover, their realizability assumption requires the function approximators to include the value/weight functions associated with a policy $\pi_n$ depending on the sample size $n$. In particular, the resulting realizability condition is difficult to interpret since there is no explicit characterization of $\pi_n$.

Chen and Jiang (2022) took a different approach, the pessimistic value learning, achieving the statistically reasonable rate $O(\epsilon^{-2})$. One of the main drawbacks of their result is, however, that the sample complexity bound blows up if the action value gap $C_{\mathrm{gap}}$ is near zero. Here, the action value gap is defined as the minimum gap in the values of the best action and the second best action, $C_{\mathrm{gap}} = \min_{s\in\mathcal{S}} \max_{a\in\mathcal{A}}\{q^*(s,a) - \max_{a'\neq a} q^*(s,a')\}$, which becomes (near) zero if there exist two actions that are (near) optimal for even one state. In addition, it only competes with the optimal policy $\pi^*$ and requires the data to cover the corresponding visitation distribution $d^{\pi^*}$. Finally, the time-horizon dependency $O(H^5)$ is a bit worse than the other results in the table.

Ozdaglar et al. (2023) showed two distinct results: one requiring a completeness-type assumption and the other requiring realizability and action-value-gap assumptions, in addition to concentrability. We included the latter to the table. Roughly speaking, their result is similar to that of Chen and Jiang (2022) except with the difference in the infinite/finite time horizons. Consequently, it also requires the action gap to be bounded away from zero. We note that their algorithm relies on a constrained LP, where the number of the constraints is equal to the size of $\mathcal{S}$, making it possibly difficult to scale to practical problems.

Uehara et al. (2023) also proposed two distinct methods: one establishes slower $O(\epsilon^{-8})$ rate with the entropy regularization method, and the other establishes a sample complexity depending on so-called the *soft action value gap*. We only shows the latter in Table 1. For the latter result, the new condition on the soft action value gap relaxes those imposed on the ordinary action value gap by Chen and Jiang (2022); Ozdaglar et al. (2023). The order of the resulting sample complexity bound is $O(\epsilon^{-2+4/\beta_{\mathrm{gap}}})$, where $\beta_{\mathrm{gap}} > 0$ corresponds to the lower-tail exponent of the distribution of the state-wise action

value gaps. We also note that these bounds are explicitly depending on the size of the action space $\mathcal{A}$, which could be a potential drawback of the entropy-based method.

Compared to these results, our sample complexity bound has the following advantages. First of all, it gives the performance guarantees even if there is no concentrable policies. Second, it only requires the model-free realizability with respect to a (fixed) worst-case optimal policy $\tilde{\pi}^*$. Third, it achieves the statistically reasonable rate $O(\epsilon^{-2})$ without any dependencies on the action value gap. Finally, it has no explicit dependency in the algorithm on the size of $\mathcal{S}$ and $\mathcal{A}$, even in the policy extraction process.

## B Rademacher Complexity and Uniform Convergence

In this section, we introduce the Rademacher complexity and its properties as well as the celebrated uniform convergence theorem. Below, let $\mathcal{Z}$ be a sample space, $p \in \Delta(\mathcal{Z})$ be a probability distribution on it, and $\mathcal{G} \subset \mathbb{R}^{\mathcal{Z}}$ be a set of functions from $\mathcal{Z}$ to $\mathbb{R}$.

**Definition B.1** (Rademacher complexity). *The Rademacher complexity of $\mathcal{G}$ with the sample size $n \geq 1$ is given by*

$$\mathfrak{R}_n(\mathcal{G}) := \mathbb{E}_{\sigma^n, z^n} \left[ \sup_{g \in \mathcal{G}} \frac{1}{n} \sum_{i=1}^n \sigma_i g(z_i) \right], \tag{23}$$

*where $\mathbb{E}_{\sigma^n, z^n}$ denotes the expectation with respect to samples $\sigma^n = \{\sigma_i\}_{i=1}^n$ and $z^n = \{z_i\}_{i=1}^n$ drawn from* $\mathrm{Uniform}^n(\{-1, +1\})$ *and $p^n$, respectively.*

**Lemma B.1** (Uniform convergence theorem). *Suppose $\|g\|_\infty \leq c$ for all $g \in \mathcal{G}$. Then, for all $\delta \in (0, 1)$, we have*

$$\sup_{g \in \mathcal{G}} \left| \frac{1}{n} \sum_{i=1}^n [g(z_i)] - \mathbb{E}[g(z_1)] \right| \leq 2\mathfrak{R}_n(\mathcal{G}) + c\sqrt{\frac{2\ln(2/\delta)}{n}}$$

*with probability $1 - \delta$ on the draw of $z^n \sim p^n$.*

*Proof.* Refer to Claim 1, Theorem 26.5, Shalev-Shwartz and Ben-David (2014) for one-side high-probability bound and apply it to both of $\sup_{g \in \mathcal{G}} \left\{ \frac{1}{n} \sum_{i=1}^n [g(z_i)] - \mathbb{E}[g(z_1)] \right\}$ and $\sup_{g \in \mathcal{G}} (-1) \left\{ \frac{1}{n} \sum_{i=1}^n [g(z_i)] - \mathbb{E}[g(z_1)] \right\}$ setting the confidence parameter to $\delta/2$. The proof is completed by taking the union of the events that these high-probability bounds do not hold. $\square$

The following is another well-known result of the Rademacher complexity.

**Lemma B.2** (Massart's lemma). *For a finite set $\mathcal{G}$, we have*

$$\mathfrak{R}_n(\mathcal{G}) \leq M(\mathcal{G})\sqrt{\frac{2\ln|\mathcal{G}|}{n}},$$

*where*

$$M(\mathcal{G}) := \sup_{g, g' \in \mathcal{G}, \, z \in \mathcal{Z}} |g(z) - g'(z)|.$$

*Proof.* Refer to Shalev-Shwartz and Ben-David (2014), Lemma 26.8. $\square$

## C Basic Properties of Regularized Lagrangian

In this section, we show basic property of regularized Lagrangian (Eq. (7)).

**Lemma C.1** (Primal Lipschitz continuity). *For all $v, v' : \mathcal{S} \times \mathcal{A} \to [0, (1-\gamma)^{-1}]$ and $f \in \mathcal{F}$, we have*

$$|K(v, f) - K(v', f)| \leq (2 + B_{\mathcal{F}}) \|v - v'\|_{1, \bar{\mu}}$$

*Proof.* Observe that by Eq. (7)

$$|K(v,f) - K(v',f)| \leq (1-\gamma)\,\|v - v'\|_{1,\mu} + \left\|f \cdot (\delta^{\mathrm{TD}}v - \delta^{\mathrm{TD}}v')\right\|_{1,\mu\beta} + (1-\gamma)\,\|v - v'\|_{1,\bar{\mu}}.$$

The second term on the RHS is further bounded as

$$\begin{aligned}
\left\|f \cdot (\delta^{\mathrm{TD}}v - \delta^{\mathrm{TD}}v')\right\|_{1,\mu\beta} &= \|f \cdot (I - \gamma\mathcal{T})(v - v')\|_{1,\mu\beta} \\
&\overset{(a)}{\leq} B_{\mathcal{F}}\,\|(I - \gamma\mathcal{T})(v - v')\|_{1,\mu\beta} \\
&\overset{(b)}{\leq} B_{\mathcal{F}}\,\|v - v'\|_{1,\bar{\mu}},
\end{aligned}$$

where (a) follows from Hölder's inequality and (b) follows from the triangle inequality. We obtain the desired result by summing up both sides of the two inequalities since $\mu \leq \bar{\mu}$. $\qquad\square$

**Lemma C.2** (Stability of minimax value against dual error). *For all $v \geq 0$ and $\pi \in \Pi_\beta$, we have*

$$K(v,f) \geq U^* - \bar{B}_{\mathcal{V}}\left\|f - \check{f}^\pi\right\|_{1,\mu\beta} - 2(1-\gamma)\,\|\tilde{v}^* - \tilde{v}^\pi\|_{1,\bar{\mu}},$$

*where $\bar{B}_{\mathcal{V}} \coloneqq \max\{1 + \gamma B_{\mathcal{V}}, B_{\mathcal{V}}\}$.*

*Proof.* Observe that

$$\begin{aligned}
K(v,f) &\overset{(a)}{\geq} K(v,f) - D_{\mathrm{V}}^\pi(v) \\
&\overset{(b)}{=} U(\pi) - \mathbb{E}_{\mu,\beta}\left[(f - \check{f}^\pi)(s,a)\,(I - \gamma\tilde{\mathcal{T}})(v - \tilde{v}^\pi)(s,a)\right] - D_{\mathrm{F}}^\pi(f) \\
&\overset{(c)}{=} U(\pi) - \mathbb{E}_{\mu,\beta}\left[(f - \check{f}^\pi)(s,a)\,(I - \gamma\tilde{\mathcal{T}})(v - \tilde{v}^\pi)(s,a)\right] \\
&\qquad - \mathbb{E}_{\mu,\beta}\left[(f - \check{f}^\pi)(s,a)\,\{\tilde{v}^\pi(s) - \tilde{q}^\pi(s,a)\}\right] \\
&\overset{(d)}{=} U(\pi) + \mathbb{E}_{\mu,\beta}\left[(f - \check{f}^\pi)(s,a)\,\left\{r(s,a) + \gamma\tilde{\mathcal{T}}v(s,a) - v(s)\right\}\right] \\
&\overset{(e)}{\geq} U(\pi) - \bar{B}_{\mathcal{V}}\left\|f - \check{f}^\pi\right\|_{1,\mu\beta},
\end{aligned}$$

where (a) follows from the nonegativity of $D_{\mathrm{V}}^\pi(v)$, (b) from Lemma E.3, (c) from $D_{\mathrm{F}}^\pi(\check{f}^\pi) = 0$, (d) from $\tilde{q}^\pi(s,a) - \gamma\tilde{\mathcal{T}}v(s,a) = r(s,a)$ and (e) from Hölder's inequality with $|r + \gamma\tilde{\mathcal{T}}v - v| \leq \bar{B}_{\mathcal{V}}$. The claim is then proved by the continuity of $U(\pi)$,

$$\begin{aligned}
U^* - U(\pi) &= (1-\gamma)\,\|\tilde{v}^* - \tilde{v}^\pi\|_{1,\mu} + \frac{(1-\gamma)^2}{2}\mathbb{E}_{\bar{\mu}}\left[(\tilde{v}^* + \tilde{v}^\pi)(\tilde{v}^* - \tilde{v}^\pi)\right] \\
&\leq (1-\gamma)\,\|\tilde{v}^* - \tilde{v}^\pi\|_{1,\mu} + (1-\gamma)\,\|\tilde{v}^* - \tilde{v}^\pi\|_{1,\bar{\mu}} \\
&\leq 2(1-\gamma)\,\|\tilde{v}^* - \tilde{v}^\pi\|_{1,\bar{\mu}}.
\end{aligned}$$

$\qquad\square$

# D  Proofs of Section 4

## D.1  Proof of Theorem 4.1

*Proof.* The claim $\tilde{\mathcal{M}} \in \mathfrak{U}$ is trivial from Definition 4.1. The other claim, $J(\pi|\tilde{\mathcal{M}}) \leq J(\pi|\mathcal{M}')$, follows from that i) $\tilde{r}(s,a) \leq r(s,a)$ with $\tilde{r}(s,a) = \mathbb{E}_{y\sim R(y|s,a)}[y]$ for all $s \in \mathcal{S}$ and $a \in \mathcal{A}$ and ii) the truncated transition probability $\tilde{T}(\cdot|s,a)$, when in conflict with $T(\cdot|s,a)$, leads to the absorbing state $\perp$, which has the lowest possible cumulative discounted value, zero. $\qquad\square$

## D.2  Proof of Corollary 4.1

*Proof.* Since the inequality is trivial from Eq. (1), we show the equality. Observe that the state-action pair $(s_t, a_t)$ at time $t \geq 0$ stays inside $\mathrm{supp}(\mu\beta)$ almost surely for all $t \geq 0$ if $\mathrm{supp}(d^\pi\pi) \subset \mathrm{supp}(\mu\beta)$. Hence, the law of the reward sequence $\{r_t\}_{t\geq 0}$ generated with $\pi$ is the same under $\mathcal{M}$ and $\tilde{\mathcal{M}}$, leading to $J(\pi|\tilde{\mathcal{M}}) = J(\pi|\mathcal{M})$. $\qquad\square$

### D.3 Proof of Lemma 4.1

*Proof.* We show denying the conclusion results in contradiction. Take a policy $\pi \in \tilde{\Pi}^*$. Let $\chi_\beta(s,a) := \mathbb{I}\{(s,a) \in \text{supp}(\beta)\}$ be the indicator function of $\text{supp}(\beta)$. Let $z^\pi(s) := \sum_a \chi_\beta(s,a)\,\pi(a|s)$ be the mass of $\pi(\cdot|s)$ inside the support of $\beta$, which satisfies $0 \le z^\pi(s) \le 1$ for all $s \in \mathcal{S}$.

Let $\pi' : \mathcal{S} \to \Delta(\mathcal{A})$ be a policy proportional to $\pi$ with the support restricted to $\text{supp}(\beta)$, i.e., $z_\pi(s)\,\pi'(a|s) = \chi_\beta(s,a)\,\pi(a|s)$ for all $s \in \mathcal{S}$ and $a \in \mathcal{A}$.

By the definitions of $\tilde{r}^\pi$ and $\tilde{\mathcal{T}}^\pi$, we now have

$$
\begin{aligned}
\tilde{r}^\pi(s) &= \sum_a \chi_\beta(s,a)\,\pi(a|s)\,r(s,a) \\
&= \sum_a z^\pi(s)\,\pi'(a|s)\,r(s,a) \\
&= z^\pi(s) \sum_a \chi_\beta(s,a)\,\pi'(a|s)\,r(s,a) \\
&= z^\pi(s)\,\tilde{r}^{\pi'}(s) \le \tilde{r}^{\pi'}(s)
\end{aligned}
$$

and

$$
\begin{aligned}
\tilde{\mathcal{T}}^\pi v(s) &= \sum_{a,s'} \pi(a|s)\,\chi_{\mu,\beta}(s,a)\,T(s'|s,a)\,v(s') \\
&= \sum_{a,s'} z^\pi(s)\,\pi'(a|s)\,T(s'|s,a)\,v(s') \\
&= z^\pi(s) \sum_{a,s'} \pi'(a|s)\,\chi_{\mu,\beta}(s,a)\,T(s'|s,a)\,v(s') \\
&= z^\pi(s)\,\tilde{\mathcal{T}}^{\pi'} v(s) \le \tilde{\mathcal{T}}^{\pi'} v(s)
\end{aligned}
$$

for all $v : \mathcal{S} \to [0,\infty)$ and $s \in \mathcal{S}$. Therefore, we have for all $s \in \mathcal{S}$

$$
\tilde{v}^*(s) = \tilde{v}^\pi(s) = (I - \gamma\tilde{\mathcal{T}}^\pi)^{-1} r^\pi(s) = \sum_{t \ge 0} (\gamma\tilde{\mathcal{T}}^\pi)^t r^\pi(s) \le \sum_{t \ge 0} (\gamma\tilde{\mathcal{T}}^{\pi'})^t r^{\pi'}(s) = \tilde{v}^{\pi'}(s),
$$

which leads to the strong optimality $\tilde{v}^{\pi'} = \tilde{v}^*$. However, we also have $\pi' \in \Pi_\beta$ by the definition, contradicting with the assumption $\tilde{\Pi}^* \cap \Pi_\beta = \emptyset$. $\qquad\square$

## E  Proofs of Section 5

### E.1  Proof of Lemma 5.1

The proof relies on two lemmas (Lemmas E.1 and E.2), where the second one is built on top of the first one. Then, Lemma 5.1 is immediately proved as a special case of Lemma E.2 with $\pi$ being restricted to $\tilde{\Pi}^*$.

Lemma E.1 gives a saddle-point decomposition of Lagrangian ignoring the offline constraint.

**Lemma E.1** (Incomplete saddle-point decomposition of Lagrangian). *For all $\pi : \mathcal{S} \to \Delta(\mathcal{A})$,*

$$
L(v,f) = J(\pi) + \sum_{s,a} (f\mu\beta - d^\pi\pi)(s,a)\,\delta^{\text{TD}} v(s,a). \tag{24}
$$

*Proof.* Comparing the LHS and the RHS, it suffices to show

$$
J(\pi) - (1-\gamma)\mathbb{E}_{p_0}[v(s)] = \sum_{s,a} d^\pi(s)\,\pi(a|s)\,\delta^{\text{TD}} v(s,a).
$$

This is seen by simplifying the RHS of the above equation as follows,

$$\sum_{s,a} d^\pi(s)\,\pi(a|s)\,\delta^{\mathrm{TD}}v(s,a) \overset{\text{(a)}}{=} \sum_s d^\pi(s)\,(I - \gamma \mathcal{T}^\pi)(v^\pi - v)$$

$$\overset{\text{(b)}}{=} \sum_s (I - \gamma \mathcal{T}^\pi_*)d^\pi(s)\,(v^\pi - v)$$

$$\overset{\text{(c)}}{=} (1 - \gamma)\sum_s p_0(s)\,(v^\pi - v)$$

$$\overset{\text{(d)}}{=} J(\pi) - (1 - \gamma)\mathbb{E}_{p_0}\left[v(s)\right].$$

Here, (a) follows from $\sum_a \pi(a|s)\,\delta^{\mathrm{TD}}v(s,a) = (r^\pi + \gamma \mathcal{T}^\pi v - v)(s)$ and $\tilde{r}^\pi = (I - \gamma\mathcal{T}^\pi)v^\pi$, (b) from $(I - \gamma\mathcal{T}^\pi_*)$ being the adjoint operator of $(I - \gamma\mathcal{T}^\pi)$, (c) from $(1 - \gamma)p_0 = (I - \gamma\mathcal{T}^\pi_*)d^\pi$, and (d) from the definition of $J(\pi)$. $\qquad\square$

Note that the second term of Eq. (24) may have no root with respect to $f$ if the data support does not cover the visitation distribution $d^\pi$, hence *incomplete*. Lemma E.2 gives a modification of Eq. (24) to fix this problem.

**Lemma E.2** (Generalized saddle-point decomposition of Lagrangian). *For all $\pi \in \Pi_\beta$, we have*

$$L(v, f) = \tilde{J}(\pi) - \mathbb{E}_{\mu,\beta}\left[(f - \tilde{f}^\pi)(s, a)\,(I - \gamma\tilde{\mathcal{T}})(v - \tilde{v}^\pi)(s, a)\right] + D^\pi_{\mathrm{V}}(v) - D^\pi_{\mathrm{F}}(f), \quad (25)$$

*where $D^\pi_{\mathrm{V}}(v) := \sum_{s\notin\mathrm{supp}(\mu)} \tilde{d}^\pi(s)\,v(s)$ and $D^\pi_{\mathrm{F}}(f) := \mathbb{E}_{\mu,\beta}\left[f(s, a)\,\{\tilde{v}^\pi(s) - \tilde{q}^\pi(s, a)\}\right].$*

*Proof.* Since Lagrangian is defined with $r(s, a)$ and $T(\cdot|s, a)$ only on the support of the offline data distribution $\mathrm{supp}(\mu\beta)$, Lemma E.1 together with the indistinguishability of $\tilde{\mathcal{M}}$ and $\mathcal{M}$ gives

$$L(v, f) = \tilde{J}(\pi) + \sum_{s,a} (f\mu\beta - \tilde{d}^\pi\pi)(s, a)\,\tilde{\delta}^{\mathrm{TD}}v(s, a),$$

where $\tilde{\delta}^{\mathrm{TD}}v := \tilde{r} + \gamma\tilde{\mathcal{T}}v - v$. The second term of the RHS of the above equation further evaluated by separating the summation to the on-support and off-support terms

$$\sum_{s,a} (f\mu\beta - \tilde{d}^\pi\pi)(s, a)\,\tilde{\delta}^{\mathrm{TD}}v(s, a)$$

$$= \left(\sum_{(s,a)\in\mathrm{supp}(\mu\beta)} + \sum_{(s,a)\notin\mathrm{supp}(\mu\beta)}\right)(f\mu\beta - \tilde{d}^\pi\pi)(s, a)\,\tilde{\delta}^{\mathrm{TD}}v(s, a)$$

$$\overset{\text{(a)}}{=} \mathbb{E}_{\mu,\beta}\left[(f - \tilde{f}^\pi)(s, a)\,\tilde{\delta}^{\mathrm{TD}}v(s, a)\right] + \sum_{(s,a)\notin\mathrm{supp}(\mu\beta)} \tilde{d}^\pi(s)\,\pi(a|s)\,v(s)$$

$$\overset{\text{(b)}}{=} -\mathbb{E}_{\mu,\beta}\left[(f - \tilde{f}^\pi)(s, a)\,(I - \gamma\tilde{\mathcal{T}})(v - \tilde{v}^\pi)(s, a)\right] - D^\pi_{\mathrm{F}}(f)$$

$$+ \left(\sum_{s\notin\mathrm{supp}(\mu),a\in\mathcal{A}} + \sum_{s\in\mathrm{supp}(\mu),a\notin\mathrm{supp}(\beta(s))}\right)\tilde{d}^\pi(s)\,\pi(a|s)\,v(s)$$

$$\overset{\text{(c)}}{=} -\mathbb{E}_{\mu,\beta}\left[(f - \tilde{f}^\pi)(s, a)\,(I - \gamma\tilde{\mathcal{T}})(v - \tilde{v}^\pi)(s, a)\right] - D^\pi_{\mathrm{F}}(f) + D^\pi_{\mathrm{V}}(v),$$

where (a) follows from that $\tilde{f}^\pi(s, a)\mu(s)\beta(a|s) = \tilde{d}^\pi(s)\pi(a|s)$ if $(s, a) \in \mathrm{supp}(\mu\beta)$ and $\tilde{\delta}^{\mathrm{TD}}v(s, a) = -v(s)$ if $(s, a) \notin \mathrm{supp}(\mu\beta)$, (b) from transforming $\tilde{r}$ in the first term, within $\tilde{\delta}^{\mathrm{TD}}$, with $\tilde{r} = (I - \gamma\tilde{\mathcal{T}})\tilde{v}^\pi + (\tilde{q}^\pi - \tilde{v}^\pi)$ and simplify the resulting $(\tilde{q}^\pi - \tilde{v}^\pi)$ with $\mathbb{E}_{a\sim\beta(a|s)}[\tilde{f}^\pi(s, a)(\tilde{q}^\pi(s, a) - \tilde{v}^\pi(s))] = 0$, and (c) from evaluating the last summation as zero with the fact $\mathrm{supp}(\mu\pi) \subset \mathrm{supp}(\mu\beta)$. $\qquad\square$

Finally, Lemma 5.1 is shown by taking $\pi$ such that $\pi \in \Pi_\beta \cap \tilde{\Pi}^*$.

## E.2 Proof of Theorem 5.1

*Proof.* First, see Appendix E.1 for the proof of Lemma 5.1. Then, recall that $(\tilde{v}^*, \tilde{f}^\pi)$ is a saddle point of $L(v, f)$ if $L(\tilde{v}^*, f) \leq L(\tilde{v}^*, \tilde{f}^\pi) \leq L(v, \tilde{f}^\pi)$ for all $v, f \geq 0$. By Eq. (6) with the nonnegativity of $D_V^\pi(v)$ and $D_F^*(f)$, it suffices to show $D_V^\pi(\tilde{v}^*) = 0$ and $D_F^*(\tilde{f}^\pi) = 0$. The former follows from $\tilde{v}^*(s) = 0$ for all $s \notin \operatorname{supp}(\mu)$ and the latter follows from $\mathbb{E}_{a \sim \beta(a|s)}[\tilde{f}^\pi(s, a) \{\tilde{v}^*(s) - \tilde{q}^*(s, a)\}] = \tilde{w}^\pi(s) \{\tilde{v}^*(s) - \mathbb{E}_{a \sim \pi(a|s)}[\tilde{q}^*(s, a)]\} = 0$ for all $\pi \in \Pi_\beta \cap \tilde{\Pi}^*$. $\qquad\square$

## E.3 Proof of Corollary 5.1

*Proof.* Note that the existence of the saddle points is always ensured since the (restricted) domains of $v$ and $f$ are all convex. Therefore, the first claim is reduced to $0 = \operatorname{argmax}_{f \geq 0} \min_{v \in \mathcal{V}_\epsilon} L(v, f)$, which is shown by Lemma 5.1 and the fact $(I - \gamma\tilde{\mathcal{T}})(v - \tilde{v}) \geq (1 - \gamma)\epsilon > 0$ for all $v \in \mathcal{V}_\epsilon$. Besides, the second claim is reduced to $0 = \operatorname{argmin}_{v \geq 0} \max_{f \in \mathcal{F}_\epsilon} L(v, f)$, which is also shown by Lemma 5.1 and the fact $\mathbb{E}_{\mu, \beta}[(f - \tilde{f}^\pi)(s, a) (I - \gamma\tilde{\mathcal{T}})(v - \tilde{v}^*)(s, a)] \geq (1 - \gamma)\epsilon\, \mathbb{E}_{\tilde{\mathcal{T}}_*^\pi \mu}[(v - \tilde{v}^*)(s)]$, which attains the minimum uniquely with $v = 0$. $\qquad\square$

## E.4 Proof of Lemma 5.2

Define

$$U(\pi) := (1 - \gamma)\mathbb{E}_\mu[\tilde{v}^\pi(s)] + \frac{(1 - \gamma^2)}{2}\|\tilde{v}^\pi\|_{2,\bar{\mu}}^2.$$

Then, Lemma 5.2 is proved as a special case of the following lemma with the restriction $\pi \in \Pi_\beta \cap \tilde{\Pi}^*$ and $U^* := (1 - \gamma)\mathbb{E}_\mu[\tilde{v}^*(s)] + \frac{(1 - \gamma^2)}{2}\|\tilde{v}^*\|_{2,\bar{\mu}}^2$.

**Lemma E.3** (Generalized saddle-point decomposition of regularized Lagrangian). *For all $\pi \in \Pi_\beta$,*

$$K(v, f) = U(\pi) - \mathbb{E}_{\mu, \beta}\left[(f - \check{f}^\pi)(s, a) (I - \gamma\tilde{\mathcal{T}})(v - \tilde{v}^\pi)(s, a)\right] + \check{D}_V^\pi(v) - D_F^\pi(f). \quad (26)$$

*Proof.* Observe that

$$v^2 = (v - \tilde{v}^\pi)^2 + 2\tilde{v}^\pi v - (\tilde{v}^\pi)^2.$$

Thus, multiplying both sides with $(1 - \gamma)^2/2$ and plugging to Eq. (7), we get

$$
\begin{aligned}
K(v, f) = {} & \frac{(1 - \gamma)^2}{2}\left(\|v - \tilde{v}^\pi\|_{2,\bar{\mu}}^2 - \|\tilde{v}^\pi\|_{2,\bar{\mu}}^2\right) \\
& + (1 - \gamma)\sum_s \{\mu(s) + (1 - \gamma)\tilde{v}^\pi(s)\,\bar{\mu}(s)\}\, v(s) + \mathbb{E}_{\mu, \beta}\left[f(s, a)\,\delta^{\mathrm{TD}}v(s, a)\right].
\end{aligned}
$$

Now, applying Lemma E.2 on the last two terms of the RHS with the formal substitution $p_0 \leftarrow \check{p}^\pi$, we have

$$
\begin{aligned}
K(v, f) = {} & \frac{(1 - \gamma)^2}{2}\left(\|v - \tilde{v}^\pi\|_{2,\bar{\mu}}^2 - \|\tilde{v}^\pi\|_{2,\bar{\mu}}^2\right) \\
& + \sum_s \check{d}^\pi(s)\,\tilde{r}^\pi(s) - \mathbb{E}_{\mu, \beta}\left[(f - \check{f}^\pi)(s, a) (I - \gamma\tilde{\mathcal{T}})(v - \tilde{v}^\pi)(s, a)\right] \\
& + \sum_{s \notin \operatorname{supp}(\mu)} \check{d}^\pi(s)\, v(s) - D_F^\pi(f) \\
= {} & \sum_s \check{d}^\pi(s)\,\tilde{r}^\pi(s) - \frac{(1 - \gamma)^2}{2}\|\tilde{v}^\pi\|_{2,\bar{\mu}} \\
& - \mathbb{E}_{\mu, \beta}\left[(f - \check{f}^\pi)(s, a) (I - \gamma\tilde{\mathcal{T}})(v - \tilde{v}^\pi)(s, a)\right] + \check{D}_V^\pi(v) - D_F^\pi(f),
\end{aligned}
$$

where $\tilde{J}(\pi)$, $\tilde{d}^\pi(s)$ and $\tilde{f}^\pi(s, a)$ in Lemma 5.1 are replaced with $\sum_s \check{d}^\pi(s)\, r^\pi(s)$, $\check{d}^\pi(s)$ and $\check{f}^\pi(s, a)$, respectively, due to the substitution. Finally, the proof is concluded by simplifying the first term on

the RHS

$$\sum_s \breve{d}^\pi(s)\, \tilde{r}^\pi(s) = (1-\gamma) \sum_s (I - \gamma \tilde{\mathcal{T}}_*^\pi)^{-1} \left\{ \mu + (1-\gamma)\tilde{v}^\pi \bar{\mu} \right\}(s)\, \tilde{r}^\pi(s)$$

$$\overset{\text{(a)}}{=} (1-\gamma) \sum_s \left\{ \mu + (1-\gamma)\tilde{v}^\pi \bar{\mu} \right\} (I - \gamma \tilde{\mathcal{T}}^\pi)^{-1} \tilde{r}^\pi(s)$$

$$\overset{\text{(b)}}{=} U(\pi) + \frac{(1-\gamma)^2}{2} \left\| \tilde{v}^\pi \right\|_{2,\bar{\mu}},$$

where (a) follows from the fact $(I - \gamma \tilde{\mathcal{T}}_*^\pi)^{-1}$ is the adjoint operator of $(I - \gamma \tilde{\mathcal{T}}^\pi)^{-1}$ and (b) is owing to $\tilde{v}^\pi = (I - \gamma \tilde{\mathcal{T}}^\pi)\tilde{r}^\pi$. $\qquad\square$

## E.5 Proof of Theorem 5.2

*Proof.* First, see Appendix E.4 for the proof of Lemma 5.2. Then, note that the third term $\breve{D}_{\mathrm{V}}^\pi(v)$ is nonegative for all $v \geq 0$ since $\breve{d}^\pi \geq 0$. Thus, we have $K(v, f) \geq U^*$ taking $f = \breve{f}^\pi$ and $K(v, f) \leq U^*$ taking $v = \tilde{v}^*$. The first claim is then proved by combining these two inequalities, in the same manner as the proof of Theorem 5.1. The second claim, the uniqueness of $\tilde{v}^*$, follows from the strong convexity of $K(\cdot, f)$. $\qquad\square$

## E.6 Proof of Lemma 5.3

*Proof.* Denote the primal excess risk function relative to $\mathcal{F}$ by

$$\varepsilon_{\mathrm{PER}}(v) := \max_{f \in \mathcal{F}} K(v, f) - \min_{v \geq 0} \max_{f \in \mathcal{F}} K(v, f)$$

and its minimizer by $\underline{v}^* := \operatorname{argmin}_{v \geq 0} \max_{f \in \mathcal{F}} K(v, f)$. Now, the strong convexity of $K(\cdot, f)$ implies the strong convexity of $\varepsilon_{\mathrm{PER}}$ and thus, for all $v : \mathcal{S} \to \mathbb{R}$,

$$\frac{(1-\gamma)^2}{2} \left\| v - \underline{v}^* \right\|_{2,\bar{\mu}}^2 \leq \varepsilon_{\mathrm{PER}}(v).$$

We utilize this bound via the triangle inequality

$$\left\| \tilde{v}_\sharp^* - \tilde{v}^* \right\|_{2,\bar{\mu}} \leq \left\| \tilde{v}_\sharp^* - \underline{v}^* \right\|_{2,\bar{\mu}} + \left\| \underline{v}^* - \tilde{v}^* \right\|_{2,\bar{\mu}} \leq \frac{\sqrt{2\varepsilon_{\mathrm{PER}}(\tilde{v}_\sharp^*)} + \sqrt{2\varepsilon_{\mathrm{PER}}(\tilde{v}^*)}}{1-\gamma}.$$

Then, each term on the RHS is bounded by

$$\varepsilon_{\mathrm{PER}}(\tilde{v}^*) \leq 2\epsilon_{\mathcal{F}}, \tag{27}$$

$$\varepsilon_{\mathrm{PER}}(\tilde{v}_\sharp^*) - \varepsilon_{\mathrm{PER}}(\tilde{v}^*) \leq (2 + B_{\mathcal{F}})\epsilon_{\mathcal{V}}, \tag{28}$$

completing the proof. The proofs of Eqs. (27) and (28) are separately given below. $\qquad\square$

*Proof of Eq. (27).* Recall that by definition

$$\varepsilon_{\mathrm{PER}}(\tilde{v}^*) = \max_{f \in \mathcal{F}} K(\tilde{v}^*, f) - \operatorname*{argmin}_{v \geq 0} \max_{f \in \mathcal{F}} K(v, f).$$

Then, on the RHS, the first term is upper bounded with $U^*$ by Theorem 5.2 and the second term (without the negative sign) is lower bounded by Lemma C.2, leading to the inequality

$$\varepsilon_{\mathrm{PER}}(\tilde{v}^*) \leq \min_{f \in \mathcal{F}} \left\{ \bar{B}_{\mathcal{V}} \left\| f - \breve{f}^\pi \right\|_{1,\mu\beta} + 2(1-\gamma) \left\| \tilde{v}^* - \tilde{v}^\pi \right\|_{1,\bar{\mu}} \right\}$$

for all $\pi \in \Pi_\beta$. The proof is completed by taking the minimum with respect to $\pi$. $\qquad\square$

*Proof of Eq. (28).* It is immediately seen from

$$\varepsilon_{\mathrm{PER}}(\tilde{v}_\sharp^*) - \varepsilon_{\mathrm{PER}}(\tilde{v}^*) = \min_{v \in \mathcal{V}} \max_{f \in \mathcal{F}} K(v, f) - \max_{f \in \mathcal{F}} K(\tilde{v}^*, f)$$

$$\leq \min_{v \in \mathcal{V}} \max_{f \in \mathcal{F}} \left\{ K(v, f) - K(\tilde{v}^*, f) \right\}.$$

$$\leq (2 + B_{\mathcal{F}}) \min_{v \in \mathcal{V}} \left\| v - \tilde{v}^* \right\|_{1,\bar{\mu}},$$

where the last inequality follows from Lemma C.1. $\qquad\square$

# F Proofs of Section 6

## F.1 Proof of Theorem 6.1

Recall that the average action value gap is given by

$$\Gamma(\pi) := \mathbb{E}_{\mu,\pi}\left[\tilde{v}^*(s) - \tilde{q}^*(s,a)\right].$$

The following lemma establishes the connection of $\Gamma(\pi_\theta)$ and $\epsilon_{\text{est}}(\theta)$.

**Lemma F.1.** *For all $\theta \in \Theta$, we have*

$$\Gamma(\pi_\theta) \leq \frac{\epsilon_{est}(\theta) + \varepsilon_{app,\Pi}(\mathcal{V},\Theta,\Xi)}{1-\gamma}.$$

*Proof.* Refer to Appendix F.2. $\qquad\square$

Then, Theorem 6.1 is proved by combining Lemma F.1 with the following lemma.

**Lemma F.2.** *For all $\pi : \mathcal{S} \to \Delta(\mathcal{A})$, we have*

$$\tilde{J}^* - \tilde{J}(\pi) \leq \|\tilde{w}^\pi\|_\infty \Gamma(\pi).$$

*Proof.* It follows directly from Hölder's inequality with the performance difference lemma for the truncated environment (Lemma F.3). $\qquad\square$

## F.2 Proof of Lemma F.1

The proof relies on the following variant of the performance difference lemma adopted for the worst-case environment $\tilde{\mathcal{M}}$.

**Lemma F.3** (Worst-case performance difference lemma). *For all $\pi : \mathcal{S} \to \Delta(\mathcal{A})$, we have*

$$\tilde{J}^* - \tilde{J}(\pi) = \mathbb{E}_{\mu,\pi}\left[\tilde{w}^\pi(s)\left\{\tilde{v}^*(s) - \tilde{q}^*(s,a)\right\}\right].$$

*Consequently, for all $\pi \in \Pi_\beta$,*

$$\tilde{J}^* - \tilde{J}(\pi) = \mathbb{E}_{\mu,\beta}\left[\tilde{f}^\pi(s,a)\left\{\tilde{v}^*(s) - \tilde{q}^*(s,a)\right\}\right].$$

*Proof.* Observe that

$$\tilde{J}^* = (1-\gamma)\sum_s p_0(s)\tilde{v}^*(s) = \sum_s \tilde{d}^\pi(s)(I - \gamma\tilde{\mathcal{T}}^\pi)\tilde{v}^*(s) = \mathbb{E}_\mu\left[\tilde{w}^\pi(s)(I - \gamma\tilde{\mathcal{T}}^\pi)\tilde{v}^*(s)\right],$$

where the second equality follows from $(I - \gamma\tilde{\mathcal{T}}_*^\pi)\tilde{d}^\pi = p_0$ and the third equality follows from $\tilde{v}^*(s) = 0$ and $\tilde{\mathcal{T}}^\pi v(s) = 0$ for all $s \notin \text{supp}(\mu)$ and $\pi : \mathcal{S} \to \Delta(\mathcal{A})$. Thus,

$$\begin{aligned}
\tilde{J}^* - J(\pi) &= \mathbb{E}_\mu\left[\tilde{w}^\pi(s)(I - \gamma\tilde{\mathcal{T}}^\pi)\tilde{v}^*(s)\right] - \mathbb{E}_\mu\left[\tilde{w}^\pi(s)\tilde{r}^\pi(s)\right] \\
&= \mathbb{E}_\mu\left[\tilde{w}^\pi(s)\left\{\tilde{v}^*(s) - \mathcal{P}^\pi\tilde{q}^*(s)\right\}\right] \\
&= \mathbb{E}_{\mu,\pi}\left[\tilde{w}^\pi(s)\left\{\tilde{v}^*(s) - \tilde{q}^*(s,a)\right\}\right]
\end{aligned}$$

where the second equality follows from $\tilde{r}^\pi + \gamma\tilde{\mathcal{T}}^\pi\tilde{v}^* = \mathcal{P}^\pi\tilde{q}^*$. This proves the first claim.

The second claim follows from the definition of $\tilde{f}^\pi$. $\qquad\square$

Let $\breve{p}^* := \mu + (1-\gamma)\tilde{v}^*\bar{\mu}$ be an alternative (unnormalized) initial state distribution and

$$\breve{f}^{\pi,*}(s,a) := (1-\gamma)\frac{(I - \gamma\mathcal{T}_*^\pi)\breve{p}^*(s)}{\mu(s)}\rho^\pi(s,a) \tag{29}$$

be the corresponding action visitation weight function. Applying Lemma F.3 to this setting, we obtain the following corollary.

**Corollary F.1.** *Define*

$$\tilde{U}(\pi) := (1-\gamma)\sum_s \breve{p}^*(s)\,\tilde{v}^\pi(s)$$

*and let $\tilde{U}^* := (1-\gamma)\sum_s \breve{p}^*(s)\,\tilde{v}^*(s)$ be its maximum. Then, we have*

$$\tilde{U}^* - \tilde{U}(\pi) = \mathbb{E}_{\mu,\beta}\left[\breve{f}^{\pi,*}(s,a)\left\{\tilde{v}^*(s) - \tilde{q}^*(s,a)\right\}\right].$$

Now we are prepared to prove Lemma F.1.

*Proof of Lemma F.1.* Let $\xi_\sharp^* \in \operatorname{argmin}_{\xi \in \Xi}\|\xi - \tilde{\xi}^*\|_{1,\mu\beta_0}$. Now, observe that

$$
\begin{aligned}
\Gamma(\pi_\theta) &= \mathbb{E}_{\mu,\pi_\theta}\left[-\tilde{\xi}^*(s,a)\right]\\
&\overset{(a)}{\le} \mathbb{E}_{\mu,\pi_\theta}\left[-\xi_\sharp^*(s,a)\right] + B_\Pi\epsilon_\Xi\\
&\overset{(b)}{\le} \frac{1}{1-\gamma}\mathbb{E}_{\mu,\pi_\theta}\left[\underline{w}_\theta(s)\left\{-\xi_\sharp^*(s,a)\right\}\right] + B_\Pi\epsilon_\Xi\\
&\overset{(c)}{\le} \frac{1}{1-\gamma}\left\{\mathbb{E}_{\mu,\beta}\left[f_\theta(s,a)\left\{-\xi_\sharp^*(s,a)\right\}\right] + D_\Xi(f_\theta;w_\theta,\pi_\theta)\right\} + B_\Pi\epsilon_\Xi\\
&\overset{(d)}{\le} \frac{1}{1-\gamma}\left\{\mathbb{E}_{\mu,\beta}\left[f_\theta(s,a)\left\{-\tilde{\xi}^*(s,a)\right\}\right] + B_{\mathcal{F}}\epsilon_\Xi + D_\Xi(f_\theta;w_\theta,\pi_\theta)\right\} + B_\Pi\epsilon_\Xi\\
&\overset{(e)}{=} \frac{1}{1-\gamma}\left\{D_{\mathrm{F}}^*(f_\theta) + \mathcal{L}_{\mathrm{X}}(\theta) + B_{\mathcal{F}}\epsilon_\Xi\right\} + B_\Pi\epsilon_\Xi && (30)
\end{aligned}
$$

where (a) follows from $\|\xi_\sharp^* - \tilde{\xi}^*\|_{1,\mu\pi_\theta} \le B_\Pi\epsilon_\Xi$, (b) from Hölder's inequality with $-\tilde{\xi}^*(s,a) \ge 0$ and $\underline{w}_\theta \ge 1-\gamma$, and (c) from Eq. (11), (d) from $\|\xi_\sharp^* - \tilde{\xi}^*\|_{1,\mu\beta} \le B_\Pi\epsilon_\Xi$, and (e) from the definition of $D_{\mathrm{F}}^*(f)$ (Lemma 5.1) and $\mathcal{L}_{\mathrm{X}}(\theta)$. In the rest of the proof, we bound $D_{\mathrm{F}}^*(f_\theta) + \mathcal{L}_{\mathrm{X}}(\theta)$ with $\epsilon_{\mathrm{est}}(\theta)$.

Fix any $\theta' \in \Theta$ and $\pi \in \Pi_\beta$. Let $\tilde{v}_\sharp^* \in \operatorname{argmin}_{v \in \mathcal{V}}\max_{f \in \mathcal{F}}K(v,f)$ and $\breve{f}_\sharp^* \in \operatorname{argmax}_{f \in \mathcal{F}}K(\tilde{v}_\sharp^*,f)$. Also let $\bar{K}^* := K(\tilde{v}_\sharp^*,\breve{f}_\sharp^*) = \min_{v \in \mathcal{V}}\max_{f \in \mathcal{F}}K(v,f)$ be the corresponding minimax value. Then, $\bar{K}^* + \mathcal{L}_{\mathrm{SP}}(\theta)$ is lower-bounded with $D_{\mathrm{F}}^*(f_\theta) - D_{\mathrm{F}}^*(f_{\theta'})$ up to an error term,

$$
\begin{aligned}
\bar{K}^* + \mathcal{L}_{\mathrm{SP}}(\theta) &= \max_{v \in \mathcal{V}}\left\{K(\tilde{v}_\sharp^*,\breve{f}_\sharp^*) - K(v,f_\theta)\right\}\\
&\overset{(a)}{\ge} K(\tilde{v}_\sharp^*,\breve{f}_\sharp^*) - K(\tilde{v}_\sharp^*,f_\theta)\\
&\overset{(b)}{\ge} K(\tilde{v}_\sharp^*,f_{\theta'}) - K(\tilde{v}_\sharp^*,f_\theta)\\
&\overset{(c)}{=} D_{\mathrm{F}}^*(f_\theta) - D_{\mathrm{F}}^*(f_{\theta'}) + \mathbb{E}_{\mu,\beta}\left[(f_\theta - f_{\theta'})(s,a)\,(I - \gamma\tilde{\mathcal{T}})(\tilde{v}_\sharp^* - \tilde{v}^*)(s,a)\right]\\
&\overset{(d)}{\ge} D_{\mathrm{F}}^*(f_\theta) - D_{\mathrm{F}}^*(f_{\theta'}) - 2B_{\mathcal{F}}\left\|\tilde{v}_\sharp^* - \tilde{v}^*\right\|_{1,\bar{\mu}}\\
&\overset{(e)}{\ge} D_{\mathrm{F}}^*(f_\theta) - D_{\mathrm{F}}^*(f_{\theta'}) - 2B_{\mathcal{F}}\,\varepsilon_{\mathrm{app,V}}(\mathcal{V},\mathcal{F}), && (31)
\end{aligned}
$$

where (a) follows from compromising the maximum with $v = \tilde{v}_\sharp^* \in \mathcal{V}$, (b) from the definition of $\breve{f}_\sharp^*$ above, (c) from Lemma 5.2, (d) from $\|f_\theta\|_\infty, \|f_{\theta'}\|_\infty \le B_{\mathcal{F}}$ and (e) from Lemma 5.3.

Moreover, let

$$\Phi(\theta') := \min_{\pi \in \Pi_\beta}\left\{\bar{B}_{\mathcal{V},\Xi}\left\|f_{\theta'} - \breve{f}^\pi\right\|_{1,\mu\beta} + 2(1-\gamma)\left\|\tilde{v}^* - \tilde{v}^\pi\right\|_{1,\bar{\mu}}\right\}$$

be the intrinsic error of $\theta'$ and fix arbitrary $\pi^* \in \Pi_\beta \cap \tilde{\Pi}^*$. Then, $\bar{K}^* + \mathcal{L}_{\mathrm{SP}}(\theta')$ is upper-bounded with $\Phi(\theta')$ up to another error term,

$$
\begin{aligned}
\bar{K}^* + \mathcal{L}_{\mathrm{SP}}(\theta') &= \bar{K}^* - U^* + U^* - \min_{v \in \mathcal{V}} K(v, f_{\theta'}) \\
&\overset{(a)}{=} K(\tilde{v}_\sharp^*, \breve{f}_\sharp^*) - K(\tilde{v}^*, \breve{f}^{\pi^*}) + U^* - \min_{v \in \mathcal{V}} K(v, f_{\theta'}) \\
&\overset{(b)}{\leq} K(\tilde{v}_\sharp^*, \breve{f}_\sharp^*) - K(\tilde{v}^*, \breve{f}_\sharp^*) + U^* - \min_{v \in \mathcal{V}} K(v, f_{\theta'}) \\
&\overset{(c)}{\leq} (2 + B_\mathcal{F}) \left\| \tilde{v}_\sharp^* - \tilde{v}^* \right\|_{1,\bar{\mu}} + \Phi(\theta') \\
&\overset{(d)}{\leq} (2 + B_\mathcal{F}) \varepsilon_{\mathrm{app,V}}(\mathcal{V}, \mathcal{F}) + \Phi(\theta'),
\end{aligned}
\tag{32}
$$

where (a) follows from Lemma 5.2, (b) from Theorem 5.2, (c) from Lemma C.1 and Lemma C.2 and (d) from Lemma 5.3.

Subtracting both sides of Eq. (31) from those of Eq. (32), we get

$$
D_{\mathrm{F}}^*(f_\theta) \leq \mathcal{L}_{\mathrm{SP}}(\theta) - \mathcal{L}_{\mathrm{SP}}(\theta') + D_{\mathrm{F}}^*(f_{\theta'}) + \Phi(\theta') + (2 + 3B_\mathcal{F}) \varepsilon_{\mathrm{app,V}}(\mathcal{V}, \mathcal{F}).
$$

which, summed with $\mathcal{L}_{\mathrm{X}}(\theta) \leq \mathcal{L}_{\mathrm{X}}(\theta) + \epsilon_{\mathrm{est}}(\theta') = \epsilon_{\mathrm{est}}(\theta) - \mathcal{L}_{\mathrm{SP}}(\theta) + \mathcal{L}(\theta')$ on both sides, yields

$$
D_{\mathrm{F}}^*(f_\theta) + \mathcal{L}_{\mathrm{X}}(\theta) \leq \epsilon_{\mathrm{est}}(\theta) + D_{\mathrm{F}}^*(f_{\theta'}) + \mathcal{L}_{\mathrm{X}}(\theta') + \Phi(\theta') + (2 + 3B_\mathcal{F}) \varepsilon_{\mathrm{app,V}}(\mathcal{V}, \mathcal{F}).
\tag{33}
$$

The remaining task is to choose $\theta' \in \Theta$ such that $D_{\mathrm{F}}^*(f_{\theta'}) + \mathcal{L}_{\mathrm{X}}(\theta') + \Phi(\theta')$ is nicely bounded. Now, observe that

$$
\begin{aligned}
D_{\mathrm{F}}^*(f_{\theta'}) &= \mathbb{E}_{\mu,\beta} \left[ f_{\theta'}(s, a) \left\{ \tilde{v}^*(s) - \tilde{q}^*(s, a) \right\} \right] \\
&\overset{(a)}{\leq} \min_{\pi \in \Pi_\beta} \left\{ \mathbb{E}_{\mu,\beta} \left[ \breve{f}^{\pi,*}(s, a) \left\{ \tilde{v}^*(s) - \tilde{q}^*(s, a) \right\} \right] + \mathbb{E}_{\mu,\beta} \left[ (f_{\theta'} - \breve{f}^\pi)(s, a) \left\{ \tilde{v}^*(s) - \tilde{q}^*(s, a) \right\} \right] \right\} \\
&\overset{(b)}{=} \min_{\pi \in \Pi_\beta} \left\{ \tilde{U}^* - \tilde{U}(\pi) - \mathbb{E}_{\mu,\beta} \left[ (f_{\theta'} - \breve{f}^\pi)(s, a) \, \tilde{\xi}^*(s, a) \right] \right\} \\
&\overset{(c)}{\leq} \min_{\pi \in \Pi_\beta} \left\{ (1 - \gamma) \left\| \tilde{v}^* - \tilde{v}^\pi \right\|_{1,\breve{p}^*} + \left\| \tilde{\xi}^* \right\|_\infty \left\| f_{\theta'} - \breve{f}^\pi \right\|_{1,\mu\beta} \right\} \\
&\overset{(d)}{\leq} \Phi(\theta'),
\end{aligned}
\tag{34}
$$

where (a) follows from $\breve{f}^{\pi,*} \geq \breve{f}^\pi$ (see Eq. (29) for the definition of $\breve{f}^{\pi,*}$), (b) from Lemma F.3, (c) from Hölder's inequality and (d) from $\breve{p}^* \leq 2\bar{\mu}$ and $\|\tilde{\xi}^*\|_\infty \leq \bar{B}_{\mathcal{V},\Xi}$. Moreover, for all $\pi \in \Pi_\beta$, we have

$$
\begin{aligned}
\mathcal{L}_{\mathrm{X}}(\theta') &= \max_{\xi \in \Xi} \left\{ \mathbb{E}_{\mu,\beta} \left[ f_\theta(s, a) \xi(s, a) \right] - \mathbb{E}_{\mu,\pi_\theta} \left[ \underline{w}_\theta(s) \xi(s, a) \right] \right\} \\
&\overset{(a)}{\leq} \max_{\xi \in \Xi} \mathbb{E}_{\mu,\beta} \left[ (f_\theta - \breve{f}^\pi)(s, a) \xi(s, a) \right] + \max_{\xi \in \Xi} \mathbb{E}_{\mu,\pi} \left[ (\breve{w}^\pi - \underline{w}_\theta)(s) \xi(s, a) \right] \\
&\quad + \max_{\xi \in \Xi} \left\{ \mathbb{E}_{\mu,\pi} \left[ \underline{w}_\theta(s) \xi(s, a) \right] - \mathbb{E}_{\mu,\pi_\theta} \left[ \underline{w}_\theta(s) \xi(s, a) \right] \right\} \\
&\overset{(b)}{\leq} B_\Xi \left\{ \left\| f_\theta - \breve{f}^\pi \right\|_{1,\mu\beta} + \left\| w_\theta - \breve{w}^\pi \right\|_{1,\mu} + B_{\underline{\mathcal{W}}} \left\| \pi_\theta - \pi \right\|_{\mathrm{TV},\mu} \right\},
\end{aligned}
\tag{35}
$$

where (a) follows from the telescoping and (b) from $\|\xi\|_\infty \leq B_\Xi$ for all $\xi \in \Xi$ and $\|\breve{w}^\pi - \underline{w}_\theta\|_{1,\mu} \leq \|\breve{w}^\pi - w_\theta\|_{1,\mu}$ since $\breve{w}^\pi \geq 1 - \gamma$. Adding both sides of Eqs. (34) and (35) and taking $\pi$ and $\theta'$ as the minimizers of Eqs. (12) and (13), respectively, we arrive at

$$
D_{\mathrm{F}}^*(f_{\theta'}) + \mathcal{L}_{\mathrm{X}}(\theta') + \Phi(\theta') \leq 2\Phi(\theta') + B_\Xi \epsilon_\Theta(\pi) \leq 3\epsilon_\Theta
\tag{36}
$$

where the last inequality follows from $\Phi(\theta') \leq \epsilon_\Theta$ (Eq. (12)). The proof is concluded by adding both sides of Eqs. (30), (33) and (36) and simplifying the error terms with Eq. (16). $\qquad\square$

## F.3 Extensions of Corollary 6.1

The suboptimality bound established by Corollary 6.1 requires the uniform boundedness of $\|\tilde{w}^{\pi_\theta}\|_\infty$. With more careful analysis, however, we can obtain policy suboptimality bounds with milder conditions. We present two of such upper bounds below.

To this end, we introduce two types of local truncated concentrability. Let $\Pi \equiv \Pi(\Theta) := \{\pi_\theta \mid \theta \in \Theta\}$ be the set of all the policy candidates.

**Definition F.1.** *Let $\tilde{C}_\epsilon := \max\{\|\tilde{w}^\pi\|_\infty \mid \tilde{J}^* - \tilde{J}(\pi) \leq \epsilon, \pi \in \Pi\}$ be the $\epsilon$-weakly local truncated concentrability ($\epsilon$-WLTC) coefficient for $\epsilon > 0$. We also define the limit WLTC coefficient as $\tilde{C}_0 := \lim_{\epsilon \to 0+} \tilde{C}_\epsilon$.*

**Definition F.2.** *Let $\tilde{c}_\epsilon := \max\{\|\tilde{w}^\pi\|_\infty \mid \max_{(s,a)\in\mathrm{supp}(\mu\pi)} \{\tilde{v}^*(s) - \tilde{q}^*(s,a)\} \leq \epsilon, \pi : \mathcal{S} \to \Delta(\mathcal{A})\}$ be the $\epsilon$-strongly local truncated concentrability ($\epsilon$-SLTC) coefficient for $\epsilon > 0$. We also define the limit SLTC coefficient as $\tilde{c}_0 := \lim_{\epsilon \to 0+} \tilde{c}_\epsilon$.*

Intuitively, both the WLTC and SLTC coefficients bound the norm of $\tilde{w}^\pi$ locally for near-optimal policies $\pi$. The difference is the ways they measure the locality: WLTC uses the policy suboptimality and SLTC uses the maximum action value gap. Note that WLTC dominates SLTC, $\tilde{c}_\epsilon \leq \tilde{C}_\epsilon \leq \tilde{C}_\infty$, if $\Pi(\Theta)$ covers the entire policy space $\Delta(\mathcal{A})^\mathcal{S}$. In general, there is no particular order between WLTC and SLTC and we just have $\tilde{C}_\epsilon \leq \tilde{C}_\infty$.

The following lemma is the foundation of our local concentrability results.

**Lemma F.4.** *For any $\pi \in \Pi$, we have*

$$\tilde{J}^* - \tilde{J}(\pi) \leq \left(1 + \frac{\tilde{c}_{\epsilon_0(\pi)}}{\tilde{c}_0}\right) \epsilon_0(\pi),$$

*where $\epsilon_0(\pi) := \sqrt{\tilde{c}_0 \Gamma(\pi)/(1-\gamma)}$.*

*Proof.* Refer to Appendix F.4. $\qquad\square$

Combining it with Lemma F.1 and discarding the non-asymptotic term for the simplicity, we get the first bound as the following corollary.

**Corollary F.2.** *For all $\theta \in \Theta$, we have*

$$\tilde{J}^* - \tilde{J}(\pi_\theta) \lesssim 2\sqrt{\frac{\tilde{c}_0}{1-\gamma} \{\epsilon_{est}(\theta) + \varepsilon_{app,\Pi}(\mathcal{V}, \Theta, \Xi)\}}, \tag{37}$$

*where $a \lesssim b$ means $\limsup_{b\to 0+} a/b \leq 1$.*

In words, Eq. (37) allows us to replace the uniform concentrability coefficient $\tilde{C}_\infty$ with the limit SLTC coefficient $\tilde{c}_0$ at the cost of the slower convergence rate due to the square root.

Moreover, a faster bound can be obtained exploiting the limit WLTC coefficient $\tilde{C}_0$ instead of $\tilde{c}_0$, leading to our second bound.

**Corollary F.3.** *For all $\theta \in \Theta$, we have*

$$\tilde{J}^* - \tilde{J}(\pi_\theta) \lesssim \frac{\tilde{C}_0}{1-\gamma} \{\epsilon_{est}(\theta) + \varepsilon_{app,\Pi}(\mathcal{V}, \Theta, \Xi)\}. \tag{38}$$

*Proof.* Observe that Corollary F.2 implies $\|\tilde{w}^{\pi_\theta}\|_\infty$ is bounded with $\tilde{C}_\epsilon$, where $\epsilon$ is taken as the RHS of Eq. (37). The claim thus follows from Theorem 6.1 with $\|\tilde{w}^{\pi_\theta}\|_\infty \leq \tilde{C}_\epsilon \to \tilde{C}_0$ as $\epsilon_{est}(\theta) + \varepsilon_{app,\Pi}(\mathcal{V}, \Theta, \Xi) \to 0$. $\qquad\square$

Similarly as Corollary F.2, in comparison to Theorem 6.1, the coefficient of the upper bound is improved from $\tilde{C}_\infty$ to $\tilde{C}_0$, meaning that asymptotically we only need the weakly local, not uniform, concentrability.

## F.4 Proof of Lemma F.4

Let us denote the set of the $\epsilon$-strongly near-optimal policies by

$$\tilde{\Pi}_\epsilon^* := \left\{ \pi : \mathcal{S} \to \Delta(\mathcal{A}) \,\middle|\, \mathrm{supp}(\pi(s)) \subset \mathrm{supp}(\tilde{\mathcal{A}}_\epsilon^*(s)), \; s \in \mathrm{supp}(\mu) \right\}, \qquad \epsilon \geq 0,$$

where $\tilde{\mathcal{A}}_\epsilon^*(s) := \{a \in \mathcal{A} \,|\, \tilde{v}^*(s) - \tilde{q}^*(s,a) \leq \epsilon\}$ is the $\epsilon$-optimal action subset for $s \in \mathcal{S}$.[7] Note that by definition $\tilde{\Pi}_0^* = \tilde{\Pi}^*$ and $\tilde{\Pi}_\epsilon^*$ is monotone nondecreasing with respect to $\epsilon$, reaching the set of the all policies with $\epsilon \geq 1/(1-\gamma)$. Then, the SLTC coefficient can be written in terms of $\tilde{\Pi}_\epsilon^*$,

$$\tilde{c}_\epsilon = \max_{\pi \in \tilde{\Pi}_\epsilon^*} \|\tilde{w}^\pi\|_\infty .$$

Now, to prove Lemma F.4, we show that there exists a strongly optimal policy $\bar{\pi} \in \tilde{\Pi}_\epsilon^*$ that approximates the target policy $\pi$ if $\Gamma(\pi)$ is small.

**Lemma F.5.** *For all $\pi : \mathcal{S} \to \Delta(\mathcal{A})$ and $\epsilon > 0$, we have*

$$\min_{\bar{\pi} \in \tilde{\Pi}_\epsilon^*} \|\pi - \bar{\pi}\|_{\mathrm{TV},\mu} \leq \frac{2\Gamma(\pi)}{\epsilon}.$$

*Proof.* Take $\pi' \in \tilde{\Pi}_\epsilon^*$ as a projection of $\pi$ onto $\tilde{\Pi}_\epsilon^*$, i.e.,

$$\pi'(a|s) = \mathbf{1}\{a \in \tilde{\mathcal{A}}_\epsilon^*(s)\}\pi(a|s) + c(s)\,\pi_0(a|s), \qquad s \in \mathcal{S},$$

with arbitrary $\pi_0 \in \tilde{\Pi}_\epsilon^*$ and $c(s) := 1 - \sum_{a \in \tilde{\mathcal{A}}_\epsilon^*(s)} \pi(a|s)$. Observe that, by the triangle inequality,

$$\|\pi - \pi'\|_{\mathrm{TV},\mu} = \mathbb{E}_\mu \sum_a |\pi(a|s) - \pi'(a|s)|$$

$$\leq \mathbb{E}_\mu \sum_a \left\{ (1 - \mathbf{1}\{a \in \tilde{\mathcal{A}}_\epsilon^*(s)\})\pi(a|s) + c(s)\,\pi_0(a|s) \right\}$$

$$= 2\mathbb{E}_\mu \left[ c(s) \right].$$

Then, we have

$$\min_{\bar{\pi} \in \tilde{\Pi}_\epsilon^*} \|\pi - \bar{\pi}\|_{\mathrm{TV},\mu} \leq 2\mathbb{E}_\mu \left[ c(s) \right] = 2\mathbb{E}_{\mu,\pi} \left[ \mathbf{1}\left\{ a \notin \tilde{\mathcal{A}}_\epsilon^*(s) \right\} \right].$$

Now, plugging

$$\mathbf{1}\left\{ a \notin \tilde{\mathcal{A}}_\epsilon^*(s) \right\} = \mathbf{1}\left\{ \tilde{v}^*(s) - \tilde{q}^*(s,a) > \epsilon \right\} \leq \frac{1}{\epsilon}\left\{ \tilde{v}^*(s) - \tilde{q}^*(s,a) \right\}$$

on the RHS, we arrive at the desired inequality

$$\min_{\bar{\pi} \in \tilde{\Pi}_\epsilon^*} \|\pi - \bar{\pi}\|_{\mathrm{TV},\mu} \leq \frac{2}{\epsilon}\mathbb{E}_{\mu,\pi}\left[\{\tilde{v}^*(s) - \tilde{q}^*(s,a)\}\right] = \frac{2}{\epsilon}\Gamma(\pi).$$

$\square$

As a corollary, we can also bound the difference of their policy values based on the SLTC coefficient.

**Corollary F.4.** *For all $\pi : \mathcal{S} \to \Delta(\mathcal{A})$ and $\epsilon > 0$, we have*

$$\min_{\bar{\pi} \in \tilde{\Pi}_\epsilon^*} \left| \tilde{J}(\pi) - \tilde{J}(\bar{\pi}) \right| \leq \frac{2\tilde{c}_\epsilon}{1-\gamma}\frac{\Gamma(\pi)}{\epsilon}.$$

*Proof.* Take arbitrary $\bar{\pi} \in \tilde{\Pi}_\epsilon^*$ and observe that

$$\tilde{d}^\pi(s) - \tilde{d}^{\bar{\pi}}(s) = (1-\gamma)\left\{(I - \gamma\tilde{\mathcal{T}}_*^\pi)^{-1} - (I - \gamma\tilde{\mathcal{T}}_*^{\bar{\pi}})^{-1}\right\}p_0(s)$$

$$= (1-\gamma)(I - \gamma\tilde{\mathcal{T}}_*^\pi)^{-1}\left\{(I - \gamma\tilde{\mathcal{T}}_*^{\bar{\pi}}) - (I - \gamma\tilde{\mathcal{T}}_*^\pi)\right\}(I - \gamma\tilde{\mathcal{T}}_*^{\bar{\pi}})^{-1}p_0(s)$$

$$= \gamma(1-\gamma)(I - \gamma\tilde{\mathcal{T}}_*^\pi)^{-1}\left(\tilde{\mathcal{T}}_*^\pi - \tilde{\mathcal{T}}_*^{\bar{\pi}}\right)(I - \gamma\tilde{\mathcal{T}}_*^{\bar{\pi}})^{-1}p_0(s)$$

$$= \gamma(I - \gamma\tilde{\mathcal{T}}_*^\pi)^{-1}\tilde{\mathcal{T}}_*\left(\mathcal{P}_*^\pi - \mathcal{P}_*^{\bar{\pi}}\right)\tilde{d}^{\bar{\pi}}(s).$$

Thus, we have

$$\left\|\tilde{w}^\pi - \tilde{w}^{\bar{\pi}}\right\|_{1,\mu} = \sum_{s \in \mathrm{supp}(\mu)}\left|\tilde{d}^\pi(s) - \tilde{d}^{\bar{\pi}}(s)\right|$$

$$= \sum_{s \in \mathrm{supp}(\mu)}\left|\gamma(I - \gamma\tilde{\mathcal{T}}_*^\pi)^{-1}\tilde{\mathcal{T}}_*\left(\mathcal{P}_*^\pi - \mathcal{P}_*^{\bar{\pi}}\right)\tilde{d}^{\bar{\pi}}(s)\right|$$

$$\overset{(a)}{\leq} \frac{\gamma}{1-\gamma}\sum_{(s,a) \in \mathrm{supp}(\mu\beta)}\left|\left(\mathcal{P}_*^\pi - \mathcal{P}_*^{\bar{\pi}}\right)\tilde{d}^{\bar{\pi}}(s,a)\right|$$

$$= \frac{\gamma}{1-\gamma}\mathbb{E}_\mu\left[\frac{\tilde{d}^{\bar{\pi}}(s)}{\mu(s)}\sum_{a \in \mathrm{supp}(\beta(s))}|\pi(a|s) - \bar{\pi}(a|s)|\right]$$

$$\overset{(b)}{\leq} \frac{\gamma\tilde{c}_\epsilon}{1-\gamma}\mathbb{E}_\mu\left[\sum_a|\pi(a|s) - \bar{\pi}(a|s)|\right]$$

$$= \frac{\gamma\tilde{c}_\epsilon}{1-\gamma}\|\pi - \bar{\pi}\|_{\mathrm{TV},\mu}, \tag{39}$$

where (a) follows from the fact $\tilde{\mathcal{T}}_*^\pi$ and $\tilde{\mathcal{T}}_*$ can be identified as non-expansive mappings of types $L^1(\mathrm{supp}(\mu)) \to L^1(\mathrm{supp}(\mu))$ and $L^1(\mathrm{supp}(\mu\beta)) \to L^1(\mathrm{supp}(\mu))$, respectively, and (b) follows from $\bar{\pi} \in \tilde{\Pi}_\epsilon^*$. Finally, observe that

$$\left|\tilde{J}(\pi) - \tilde{J}(\bar{\pi})\right| = \left|\mathbb{E}_\mu\left[(\tilde{w}^\pi\tilde{r}^\pi - \tilde{w}^{\bar{\pi}}\tilde{r}^{\bar{\pi}})(s)\right]\right|$$

$$\overset{(a)}{\leq} \mathbb{E}_\mu\left[\left|(\tilde{w}^\pi - \tilde{w}^{\bar{\pi}})\tilde{r}^\pi\right|(s) + \left|\tilde{w}^{\bar{\pi}}(\tilde{r}^\pi - \tilde{r}^{\bar{\pi}})\right|(s)\right]$$

$$\overset{(b)}{\leq} \left\|\tilde{w}^\pi - \tilde{w}^{\bar{\pi}}\right\|_{\mu,1} + \tilde{c}_\epsilon\|\pi - \bar{\pi}\|_{\mathrm{TV},\mu}$$

$$\overset{(c)}{\leq} \frac{\tilde{c}_\epsilon}{1-\gamma}\|\pi - \bar{\pi}\|_{\mathrm{TV},\mu},$$

where (a) follows from the triangle inequality, (b) from the fact $|\tilde{r}(s,a)| \leq 1$ and (c) from Eq. (39). The proof is concluded by applying Lemma F.5 on the RHS. $\qquad\square$

We now arrive at the following corollary, which immediately implies Lemma F.4 as a special case with the substitution $\epsilon = \epsilon_0(\pi)$.

**Corollary F.5.** *For all $\epsilon > 0$, we have*

$$\tilde{J}^* - \tilde{J}(\pi) \leq \epsilon + \frac{2\tilde{c}_\epsilon}{1-\gamma}\frac{\Gamma(\pi)}{\epsilon}.$$

*Proof.* By Lemma F.3, we have

$$\tilde{J}^* - \tilde{J}(\bar{\pi}) = \mathbb{E}_{\mu,\pi}\left[\tilde{w}^\pi(s)\{\tilde{v}^*(s) - \tilde{q}^*(s,a)\}\right] \leq \epsilon\mathbb{E}_\mu\left[\tilde{w}^\pi(s)\right] \leq \epsilon$$

for all $\bar{\pi} \in \tilde{\Pi}_\epsilon^*$. Thus, decomposing the suboptimality as

$$\tilde{J}^* - \tilde{J}(\bar{\pi}) \leq \tilde{J}^* - \tilde{J}(\bar{\pi}) + \left|\tilde{J}(\pi) - \tilde{J}(\bar{\pi})\right|$$

and applying Corollary F.4 results in the desired result. $\qquad\square$

### F.5 Proof of Corollary 6.2

*Proof.* The celebrated uniform convergence theorem (Lemma B.1) with the boundedness $\left|\hat{\mathcal{L}}_z(\theta; v, \xi)\right| \leq B_{\text{all}}$ implies that, for all $\delta \in (0, 1)$,

$$\max_{\theta \in \Theta, v \in \mathcal{V}, \xi \in \Xi} \left|\hat{\mathcal{L}}(\theta; v, \xi) - \mathbb{E}\left[\mathcal{L}(\theta; v, \xi)\right]\right| \leq 2\mathfrak{R}_n(\mathcal{H}) + B_{\text{all}}\sqrt{\frac{\ln(2/\delta)}{2n}} \tag{40}$$

with probability $1 - \delta$. Now, by definition, $\epsilon_{\text{est}}(\theta)$ is uniformly approximated with $\hat{\epsilon}_{\text{est}}(\theta)$ up to as twice as the statistical error given by Eq. (40). Plugging this into Eq. (18), we obtain the desired bound. $\square$

